# OT4P: Unlocking Effective Orthogonal Group Path for Permutation Relaxation

**Yaming Guo[1,4], Chen Zhu[2]\*, Hengshu Zhu[3,4]\*, Tieru Wu[1]\***
[1]School of Artificial Intelligence, Jilin University
[2]School of Management, University of Science and Technology of China
[3]Computer Network Information Center, Chinese Academy of Sciences
[4]The Hong Kong University of Science and Technology (Guangzhou)
{yamingguo98,zc3930155,zhuhengshu}@gmail.com,wutr@jlu.edu.cn

## Abstract

Optimization over permutations is typically an NP-hard problem that arises extensively in ranking, matching, tracking, etc. Birkhoff polytope-based relaxation methods have made significant advancements, particularly in penalty-free optimization and probabilistic inference. Relaxation onto the orthogonal group offers unique potential advantages such as a lower representation dimension and preservation of inner products; however, equally effective approaches remain unexplored. To bridge the gap, we present a temperature-controlled differentiable transformation that maps unconstrained vector space to the orthogonal group, where the temperature, in the limit, concentrates orthogonal matrices near permutation matrices. This transformation naturally implements a parameterization for the relaxation of permutation matrices, allowing for gradient-based optimization of problems involving permutations. Additionally, by deriving a re-parameterized gradient estimator, this transformation also provides efficient stochastic optimization over the latent permutations. Extensive experiments involving the optimization over permutation matrices validate the effectiveness of the proposed method.

## 1 Introduction

Permutation refers to the reordering of elements within a finite set, commonly encountered in problems involving bijections between two equally sized sets [17, 42, 15, 12]. A permutation of $n$ elements can be denoted by an $n \times n$ permutation matrix, which is a square binary matrix that has exactly one entry of $1$ in each row and each column, with all other entries being $0$. We denote the set of all $n$-order permutation matrices as $\mathcal{P}_n := \{P \in \{0,1\}^{n \times n} \mid \sum_i P_{i,j} = 1, \sum_j P_{i,j} = 1 \ (\forall i, j)\}$. This work considers optimization over permutation matrices:

$$\min_{P \in \mathcal{P}_n} f(P). \tag{1}$$

Due to the combinatorial nature of permutation matrices, the cardinality of the set $\mathcal{P}_n$ grows factorially with the dimension $n$, typically rendering the problem NP-hard [26]. From a theoretical perspective, one of the most renowned special cases of Equation (1) is the quadratic assignment problem, which has attracted extensive research [37, 44]. In practical terms, Equation (1) also arises extensively in various machine learning tasks, including ranking [20, 75, 67, 68], matching [2], tracking [43], etc.

Previous studies have proposed relaxing permutation matrices into continuous spaces, including the convex hull of permutation matrices—the Birkhoff polytope [15, 42]—and their embeddings in a differentiable manifold—the orthogonal group [72, 27]. Recently, relaxation methods involving the Birkhoff polytope have made significant advancements, particularly in penalty-free optimization

---

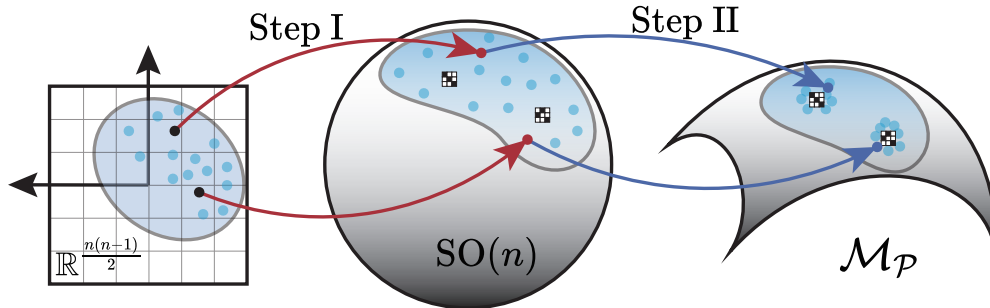

Figure 1: Illustration of OT4P with colored dots to help visualize the transformation. In the limit of temperature, the orthogonal matrices obtained from OT4P converge near the permutation matrices.

and probabilistic inference [59, 43, 49]. As a notable example, Mena et al. [49] utilize the Sinkhorn operator [61] to transform matrices into the Birkhoff polytope, bringing them closer to permutation matrices under temperature control. This approach avoids introducing penalty terms and supports variational inference.

However, providing equally good relaxation methods within the orthogonal group remains an unexplored area. Indeed, relaxation onto the orthogonal group offers several unique potential advantages, such as: i) a lower representation dimension ($\frac{n(n-1)}{2}$) compared to the Birkhoff polytope ($(n-1)^2$), leading to a smaller search space; ii) the orthogonal matrix preserve the inner product of vectors, which is useful for tasks requiring the maintenance of geometric structures. In light of the above advantages, this work aims to develop an effective method for relaxing the permutation matrices onto the orthogonal group, with a particular focus on:

- Flexibility: can control the degree of approximation to permutation matrices.
- Simplicity: does not rely on additional penalty terms.
- Scalability: enables learning the latent variable model with permutations.

In this paper, we present ***O***rthogonal ***G***roup-based ***T***ransformation ***for P***ermutation ***R***elaxation (OT4P), a temperature-controlled differentiable transformation. OT4P maps unconstrained vector space to the orthogonal group, where the temperature, in the limit, concentrates orthogonal matrices near permutation matrices. As illustrated in Figure 1, OT4P involves two steps: I) map a vector (●) to an orthogonal matrix (●) utilizing the Lie exponential; II) move the orthogonal matrix (●) along the geodesic, controlled by temperature, to another orthogonal matrix (●), making it nearer to the closest permutation matrix (▦ or ▦). OT4P naturally implements a parameterization for the relaxation of permutation matrices, allowing for gradient-based optimization of problems involving permutations. In addition, OT4P, combined with the re-parameterization trick, provides stochastic optimization over the latent permutations.

In summary, our main contributions are as follows:

1. We present OT4P, a differentiable transformation for relaxing permutation matrices onto the orthogonal group, characterized by its flexibility, simplicity, and scalability (Section 3.1).

2. We use OT4P to implement a parameterization for the relaxation of permutation matrices, which has the advantages of not altering the original problem, not complicating the original problem, and an efficient optimization process (Section 3.2).

3. We derive a gradient estimator using OT4P and the re-parameterization trick, providing an efficient tool for stochastic optimization over latent permutations (Section 3.3).

4. We validate the effectiveness of the proposed method through extensive experiments involving the optimization of permutation matrices, including finding mode connectivity, inferring neuron identities, and solving permutation synchronization (Section 4).

## 2 Preliminaries

In this section, we give a brief overview of the Riemannian geometry [38] and the Lie group theory [21] involved, with a more comprehensive version available in Appendix B.

An $n$-dimensional *manifold* $\mathcal{M}$ is a space that can be locally approximated by a Euclidean space $\mathbb{R}^n$, where each point $x \in \mathcal{M}$ possesses a *tangent space* $T_x\mathcal{M}$ as a first-order local approximation of $\mathcal{M}$ around $x$. The *Riemannian metric* is a collection $m := \{m_x \mid x \in \mathcal{M}\}$ of inner products $m_x(\cdot, \cdot) : T_x\mathcal{M} \times T_x\mathcal{M} \to \mathbb{R}$, which may define distances on the manifold. A *geodesic* is a smooth curve that the tangent vector is parallel transported along the curve w.r.t. the Levi-Civita connection.

A *Lie group* $G$ is a differentiable manifold equipped with differentiable group operations, whose tangent space at the identity $e$ is the *Lie algebra* $\mathfrak{g}$. For each $g \in G$, there exist diffeomorphisms (Definition 2) given by the *left translation* $L_g(x) := gx$ ($\forall x \in G$), which lead to a vector space isomorphism (Definition 1) that relates the tangent space $T_gG$ to the Lie algebra $\mathfrak{g}$, i.e., $(\mathrm{d}L_g)_e : \mathfrak{g} \to T_gG$. Analogously, one can introduce the *right translation*: $R_g(x) := xg$ ($\forall g, x \in G$). A Riemannian metric $m$ on Lie group $G$ is called *left-invariant* (*right-invariant*) if it renders each left (right) translation an isometry (Definition 3), allowing us to associate neighborhoods of the identity $e$ with any point $g \in G$ using left (right) translation. If a metric on the Lie group is both left and right invariant, it is termed the *bi-invariant* metric.

The Lie group we are interested in is the *orthogonal group* $\mathrm{O}(n)$, which consists of all $n \times n$ orthogonal matrices $O$ satisfying $O^\top O = OO^\top = I$. We equip $\mathrm{O}(n)$ with the canonical metric, a bi-invariant metric defined as $\langle A, B \rangle_\mathrm{F} := \mathrm{trace}(A^\top B)$, where $\langle \cdot, \cdot \rangle_\mathrm{F}$ is the Frobenius inner product and $\mathrm{trace}(\cdot)$ is the trace of a matrix. The subset of $\mathrm{O}(n)$ with determinant $+1$ forms a subgroup known as the *special orthogonal group*, denoted by $\mathrm{SO}(n) := \{O \in \mathbb{R}^{n \times n} \mid O^\top O = I, \det O = +1\}$. The Lie algebra $\mathfrak{so}(n)$ of the Lie group $\mathrm{SO}(n)$ comprises $n \times n$ skew-symmetric matrices, expressed as $\mathfrak{so}(n) := \{A \in \mathbb{R}^{n \times n} \mid A^\top = -A\}$. The *Lie exponential* $\mathrm{expm}(\cdot)$, coinciding with the *matrix exponential* in the context of the matrix Lie group, maps elements in $\mathfrak{so}(n)$ to $\mathrm{SO}(n)$, defined by

$$\mathrm{expm}(A) := I + \sum_{k=1}^{\infty} \frac{A^k}{k!}. \tag{2}$$

The series in Equation (2) converges for all matrices $A$. The local inverse function of the matrix exponential is supposed to be the *matrix logarithm*, which is defined as follows:

$$\mathrm{logm}(A) := \sum_{k=1}^{\infty} (-1)^{k+1} \frac{(A - I)^k}{k}. \tag{3}$$

The series in Equation (3) converges whenever $\|A - I\|_\mathrm{F} < 1$, where $\|A\|_\mathrm{F} := \sqrt{\langle A, A \rangle_\mathrm{F}}$ represents the Frobenius norm induced by the Frobenius inner product.

## 3 Relaxing permutation on orthogonal group

In Section 3.1, we introduce the two steps of the proposed `OT4P` and analyze its key properties. Then, in Section 3.2, we demonstrate how to use `OT4P` to implement a parameterization for the relaxation of permutation matrices, emphasizing the advantages of such a parameterization. Finally, in Section 3.3, we provide efficient stochastic optimization over the latent permutations using `OT4P` and the re-parameterization trick.

### 3.1 The proposed `OT4P` transformation

The proposed `OT4P` comprises two steps: I) map a point in the vector space to an orthogonal matrix; II) move the orthogonal matrix along the geodesic under temperature control, bringing it nearer to the closest permutation matrix. We summarize the pseudo-code of `OT4P` in Algorithm 1.

**Step I**

Consider an unconstrained vector space $\mathbb{R}^{\frac{n(n-1)}{2}}$. For a vector $a \in \mathbb{R}^{\frac{n(n-1)}{2}}$, we can fill it into an upper triangular $n \times n$ matrix with zero in the diagonal. For example, in the case of $n = 3$:

$$[a_1, a_2, a_3] = a \rightleftharpoons A = \begin{pmatrix} 0 & a_1 & a_2 \\ 0 & 0 & a_3 \\ 0 & 0 & 0 \end{pmatrix}.$$

**Algorithm 1** OT4P

---

**Input:** Input matrix $A \in \mathbb{R}^{n \times n}$, hyperparameters $\tau \in (0, 1]$, $B \in \mathrm{SO}(n)$
 1: Map $A$ to an orthogonal matrix: $O = \phi(A)$ ▷ Defined in Equation (6)
 2: Shift $O$ to handle boundary issues: $O = BO$ ▷ Details in Appendix C
 3: Find the closest permutation matrix: $P = \rho(O)$ ▷ Defined in Equation (7)
 4: **if** $\det(P) = -1$ **then**
 5:      Set $D = \mathrm{diag}(\{1, \ldots, 1, -1\})$ ▷ Extension to odd permutations, see Appendix D
 6: **else**
 7:      Set $D = I$
 8: **end if**
 9: Move $O$ toward $P$ along the geodesic controlled by $\tau$: $\widetilde{O} = \psi_\tau(O)$ ▷ Defined in Equation (11)
**Output:** The resulting orthogonal matrix $\widetilde{O}$ converge near the permutation matrix $P$

---

In the following, we employ matrices $A \in \mathbb{R}^{n \times n}$ rather than vectors $a$ to represent the elements in $\mathbb{R}^{\frac{n(n-1)}{2}}$. A skew-symmetric matrix is uniquely determined by $\frac{n(n-1)}{2}$ scalars, i.e., the entries above the main diagonal. Therefore, there exists an isomorphism between the vector space $\mathbb{R}^{\frac{n(n-1)}{2}}$ and the Lie algebra $\mathfrak{so}(n)$ formed by skew-symmetric matrices, given by

$$\begin{aligned} \alpha : \mathbb{R}^{\frac{n(n-1)}{2}} &\to \mathfrak{so}(n) \\ A &\mapsto A - A^\top. \end{aligned} \tag{4}$$

As mentioned in Section 2, we can use the matrix exponential (Lie exponential) $\mathrm{expm}(\cdot)$ to map the Lie algebra $\mathfrak{so}(n)$ to the Lie group, i.e., special orthogonal group $\mathrm{SO}(n)$:

$$\begin{aligned} \beta : \mathfrak{so}(n) &\to \mathrm{SO}(n) \\ A &\mapsto \mathrm{expm}(A). \end{aligned} \tag{5}$$

Combining Equation (4) and Equation (5), we can map the unconstrained vector space $\mathbb{R}^{\frac{n(n-1)}{2}}$ to the special orthogonal group $\mathrm{SO}(n)$, denoted as

$$\begin{aligned} \phi : \mathbb{R}^{\frac{n(n-1)}{2}} &\to \mathrm{SO}(n) \\ A &\mapsto \mathrm{expm}(A - A^\top). \end{aligned} \tag{6}$$

This mapping belongs to the classical category in Lie group theory and serves as an efficient solution for addressing orthogonal constraints in the field of machine learning [40, 53, 48]. Similarly to Lezcano Casado [39], we present the important properties of the mapping $\phi(\cdot)$ below.

**Theorem 1.** *The mapping $\phi(\cdot)$ is differentiable, surjective, and it is injective on the domain $\mathcal{U} := \{A \in \mathbb{R}^{\frac{n(n-1)}{2}} \mid \mathrm{Im}\, \lambda_k(A - A^\top) \in (-\pi, \pi), \forall k\}$ with $\lambda_k(\cdot)$ the eigenvalues. Additionally, the set $\mathrm{SO}(n) \setminus \phi(\mathcal{U})$ has a zero Lebesgue measure in $\mathrm{SO}(n)$.*

The theorem indicates that each orthogonal matrix in $\mathrm{SO}(n)$ can be represented by a vector in $\mathbb{R}^{\frac{n(n-1)}{2}}$, with each representation being uniquely defined within set $\mathcal{U}$, provided it exists there. However, permutation matrices may include $-1$ as one of their eigenvalues (see Figure 4), with their corresponding representations precisely lying on the boundary of $\mathcal{U}$. If the optimal solution to Equation (1) is a permutation matrix with an eigenvalue of $-1$, it may lead the optimization path to deviate from $\mathcal{U}$. To counter this, we propose shifting the boundary of $\mathcal{U}$ to other eigenvalues by left-multiplying the result of Equation (6) with an orthogonal matrix $B \in \mathrm{SO}(n)$. Theoretically, the left translation $L_B(O) := BO$ $(\forall O \in \mathrm{SO}(n))$ creates a diffeomorphism (Definition 2) on $\mathrm{SO}(n)$, where the representation of the permutation matrix $P$ in $\mathcal{U}$ is changed from $\mathrm{logm}(P)$ to $\mathrm{logm}(B^\top P)^2$. We have empirically observed that the left translation $L_B$ effectively relocates the majority of permutation matrices' representations into the interior of $\mathcal{U}$. More discussion can be found in Appendix C.

**Step II**

Given an orthogonal matrix $O \in \mathrm{SO}(n)$, we would like to move it toward the closest permutation matrix along the geodesic. To achieve this, we first need to find the permutation matrix $P \in \mathcal{P}_n$ closest to $O$, which can be expressed as

$$\rho(O) := \arg\max_{P \in \mathcal{P}_n} \langle P, O \rangle_{\mathrm{F}}, \tag{7}$$

where $\langle A, B \rangle_{\mathrm{F}} = \mathrm{trace}(A^\top B)$ denotes the Frobenius inner product. Equation (7) is a linear assignment problem that can be solved in cubic time using the Hungarian algorithm [36], with further details available in the Appendix E.

Once $P$ is found, we can move $O$ towards $P$ along the geodesic $OP$. In the Lie group, utilizing the mapping $\mathrm{logm}(\cdot)$, movement along the geodesic can be transformed into a more manageable movement on the Lie algebra. However, since $O$ or $P$ may be far from the identity matrix $I$, the convergence speed of the series expansion of $\mathrm{logm}(\cdot)$ may be slow or even fail to converge.

We propose to carry out the above process in the tangent space $T_P\mathrm{SO}(n)$ rather than in the Lie algebra $\mathfrak{so}(n)$. Due to the bi-invariant metric $\langle \cdot, \cdot \rangle_{\mathrm{F}}$ equipped on $\mathrm{SO}(n)$, the left translation $L_P(O) = PO$ $(\forall O \in \mathrm{SO}(n))$ establishes an isometry (Definition 3) between the neighborhoods of $I$ and $P$, and its derivative $(\mathrm{d}L_P)_e : \mathfrak{so} \to T_P\mathrm{SO}(n)$ provides an isomorphism (Definition 1) between the Lie algebra $\mathfrak{so}(n)$ and the tangent space $T_P\mathrm{SO}(n)$. Hence, we first push $O \in \mathrm{SO}(n)$ into the neighbor-

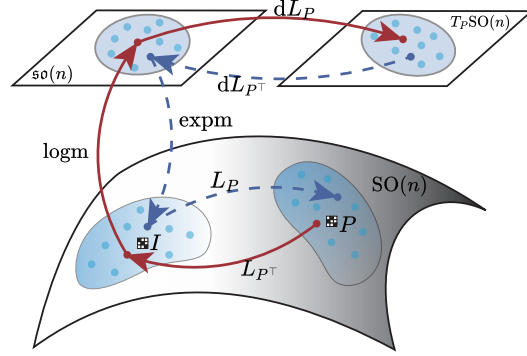

Figure 2: Illustration of the mappings $\mathrm{logm}_P$ and $\mathrm{expm}_P$. The left translation $L_P$ establishes an isometry between the neighborhoods of $I$ and $P$, and its derivative $(\mathrm{d}L_P)_e$ provides an isomorphism between $\mathfrak{so}(n)$ and $T_P\mathrm{SO}(n)$.

hood of $I$, then map it to the Lie algebra $\mathfrak{so}(n)$ using $\mathrm{logm}(\cdot)$, and finally pull the result into the tangent space $T_P\mathrm{SO}(n)$. In this way, we define the logarithm map at $P$ as

$$\begin{aligned} \mathrm{logm}_P &: \mathrm{SO}(n) \to T_P\mathrm{SO}(n) \\ O &\mapsto P\,\mathrm{logm}(P^\top O). \end{aligned} \tag{8}$$

Equation (8) maps $O \in \mathrm{SO}(n)$ near $P$ to the tangent space $T_P\mathrm{SO}(n)$, and its convergence domain also concentrates near $P$. Similarly, we can define its local inverse mapping:

$$\begin{aligned} \mathrm{expm}_P &: T_P\mathrm{SO}(n) \to \mathrm{SO}(n) \\ A &\mapsto P\,\mathrm{expm}(P^\top A). \end{aligned} \tag{9}$$

Equation (9) maps $A \in T_P\mathrm{SO}(n)$ onto the special orthogonal group $\mathrm{SO}(n)$ located near $P$. With the aforementioned tools, we can easily move orthogonal matrix $O$ toward its closest permutation matrix $P$ by interpolation. Specifically, we map $P$ and $O$ to the tangent space $T_P\mathrm{SO}(n)$ for linear interpolation, and then map the interpolation result back to $\mathrm{SO}(n)$, given as

$$\begin{aligned} \widetilde{O} &= P\,\mathrm{expm}(P^\top \left[ \tau P\,\mathrm{logm}(P^\top O) + (1-\tau)P\,\mathrm{logm}(P^\top P) \right]) \\ &= P\,\mathrm{expm}(P^\top \left[ \tau P\,\mathrm{logm}(P^\top O) \right]) \\ &= P\,\mathrm{expm}(\tau\,\mathrm{logm}(P^\top O)) \\ &= P(P^\top O)^\tau. \end{aligned} \tag{10}$$

The second equation stems from the fact that $\mathrm{logm}(I) = \mathbf{0}$, and the last equation follows from $A^\tau = \mathrm{expm}(\tau\,\mathrm{logm}(A))$ when $\|A - I\|_{\mathrm{F}} < 1$. The temperature parameter $\tau \in (0, 1]$ is used to control the degree to which the resulting orthogonal matrix $\widetilde{O}$ approaches $P$. It is clear that $\lim_{\tau \to 0^+} \|\widetilde{O} - P\|_{\mathrm{F}} = 0$.

**Remark on odd permutations.** The attentive reader may notice that Equation (10) presupposes $P \in \mathrm{SO}(n)$, which works only for $P$ corresponds to even permutations. However, we can readily extend it to cases where $P$ corresponds to odd permutations. Firstly, we solve $\arg\min_{\widehat{P} \in \mathrm{SO}(n)} \|\widehat{P} - P\|_{\mathrm{F}}^2$ to identify an agent of $P$ within $\mathrm{SO}(n)$, which admits an analytical solution $\widehat{P} = PD$ with $D = \mathrm{diag}(\{1, \ldots, 1, -1\})$. Then, by substituting $P$ with $\widehat{P}$ in Equation (10), $O$ is moved toward $\widehat{P}$ to obtain $\widehat{O}$. Finally, we right-multiply $\widehat{O}$ by $D^{\top}$ to map it to the neighborhood of $P$, resulting in $\widetilde{O} = \widehat{O}D^{\top}$. Essentially, the neighborhoods of $P$ and its agent $\widehat{P}$ are linked through an isometry, specifically the right translation $R_D(O) := OD$ ($\forall O \in \mathrm{O}(n)$). Consequently, when $\widehat{O}$ is moved close enough to $\widehat{P}$, the resulting orthogonal matrix $\widetilde{O}$ will also be sufficiently close to $P$. Please refer Appendix D for more theoretical details.

Let $\mathcal{S}_P$ denote the set of orthogonal matrices in $\mathrm{SO}(n)$ whose closest permutation matrix is $P$, and let $\mathcal{S}_P'$ represent its image obtained through Equation (10). Equation (10) actually transforms the submanifold $\mathcal{S}_P$ into a new submanifold $\mathcal{S}_P'$ that is closer to the permutation matrix $P$. We define the manifold consisting of all images as $\mathcal{M}_{\mathcal{P}} := \{\mathcal{S}_P' \subset \mathrm{O}(n) \mid P \in \mathcal{P}_n\}$. Consequently, the special orthogonal group $\mathrm{SO}(n)$ is mapped to a manifold $\mathcal{M}_{\mathcal{P}}$ that tightly wraps around the permutation matrices, which can be more formally expressed as:

$$\psi_\tau : \mathrm{SO}(n) \to \mathcal{M}_{\mathcal{P}}$$
$$O \mapsto \rho(O)D\left([\rho(O)D]^{\top}O\right)^{\tau}D^{\top}. \tag{11}$$

The aforementioned mapping covers all cases, where $D = \mathrm{diag}(\{1, \ldots, 1, -1\})$ for odd permutations and $D = I$ for even permutations. In Figure 5, we provide a visualization of the results of $\psi_\tau(\cdot)$ as the temperature parameter $\tau$ varies. The mapping $\psi_\tau(\cdot)$ is meaningless at points where $\rho(\cdot)$ w.r.t. Equation (7) is discontinuous. It is important to note that $\rho(\cdot)$ is a piecewise constant function, changing only at points where multiple permutation matrices are equidistant. The following theorem presents key properties of the mapping $\psi_\tau(\cdot)$.

**Theorem 2.** *The mapping $\psi_\tau(\cdot)$ is differentiable, surjective, and injective on each submanifold $\mathcal{S}_P$. Additionally, the set of meaningless points for $\psi_\tau(\cdot)$ has a zero Lebesgue measure in $\mathrm{SO}(n)$.*

The theorem shows that any point in the relaxation manifold $\mathcal{M}_{\mathcal{P}}$ of permutation matrices can be uniquely identified by an orthogonal matrix in the special orthogonal group $\mathrm{SO}(n)$, where the set of meaningless elements (i.e., not mapped any point in $\mathcal{M}_{\mathcal{P}}$) can be disregarded. Using the composite mapping $\psi_\tau \circ \phi$, we create a one-to-one correspondence between $\mathcal{U} := \{A \in \mathbb{R}^{\frac{n(n-1)}{2}} \mid \mathrm{Im}\,\lambda_k(A - A^{\top}) \in (-\pi, \pi), \forall k\}$ and $\mathcal{M}_{\mathcal{P}}$, except for points associated with zero measure sets in $\mathrm{SO}(n)$ that are either not representable by $\phi(\cdot)$ or are meaningless for $\psi_\tau(\cdot)$. In other words, points in the manifold $\mathcal{M}_{\mathcal{P}}$, which tightly wraps around the permutation matrices, can almost be represented one-to-one by points in $\mathcal{U}$ that lie in the unconstrained vector space.

## 3.2 Parameterization for gradient-based optimization

This section demonstrates how to use `OT4P` to implement a parameterization for the relaxation of permutation matrices, thereby allowing gradient-based optimization for Equation (1). More importantly, we present three advantages of this parameterization, making it a reasonable solution.

Recalling the manifold $\mathcal{M}_{\mathcal{P}}$ obtained from Equation (11), which converges around the permutation matrices controlled by the temperature parameter $\tau$. We first relax Equation (1) into an optimization problem on the manifold $\mathcal{M}_{\mathcal{P}}$:

$$\min_{O \in \mathcal{M}_{\mathcal{P}}} f(O). \tag{12}$$

Using the composite mapping $\psi_\tau \circ \phi$, we transform the constrained optimization problem on the manifold $\mathcal{M}_{\mathcal{P}}$ into an unconstrained optimization problem in the vector space $\mathbb{R}^{\frac{n(n-1)}{2}}$:

$$\min_{A \in \mathbb{R}^{\frac{n(n-1)}{2}}} f(\psi_\tau \circ \phi(A)). \tag{13}$$

For the aforementioned optimization problem, we can employ standard optimization techniques, such as SGD and Adam algorithms [58], to approximate the solution. Below, we thoroughly discuss the three advantages brought about by the parameterization of `OT4P`.

**The surjectivity does not alter the original problem.** The surjectivity of the mapping $\psi_\tau \circ \phi$ implies that every point in the manifold $\mathcal{M}_\mathcal{P}$ has at least one corresponding pre-image in the vector space $\mathbb{R}^{\frac{n(n-1)}{2}}$. This guarantees that, during the optimization process, any point within the manifold $\mathcal{M}_\mathcal{P}$ can be reached, thereby preventing the overlooking of any potential solutions to Equation (12). In particular, if we find a solution $A$ while dealing with Equation (13), we can solve Equation (12) by mapping $O = \psi_\tau \circ \phi(A)$.

**The injectivity does not complicate the original problem.** If the optimization stays within $\mathcal{U} := \{A \in \mathbb{R}^{\frac{n(n-1)}{2}} \mid \mathrm{Im}\, \lambda_k(A - A^\top) \in (-\pi, \pi), \forall k\}$, the mapping $\psi_\tau \circ \phi$ map different elements in $\mathcal{U}$ to different elements on $\mathcal{M}_\mathcal{P}$. This means that each update in $\mathcal{U}$ results in a unique outcome in $\mathcal{M}_\mathcal{P}$, thereby reducing unnecessary redundant searches. Furthermore, the mapping $\psi_\tau \circ \phi$ does not introduce spurious local minima, as each local minima in $\mathcal{M}_\mathcal{P}$ creates a single local minima in $\mathcal{U}$.

**The efficient optimization process.** At first glance, the mapping $\psi_\tau \circ \phi$ involves matrix exponential and matrix power, which might demand substantial computational resources during the optimization process. Lezcano-Casado and Martınez-Rubio [40] has proposed a cheap method for computing matrix exponential $\mathrm{expm}(\cdot)$ and its gradient, thanks to the efficient utilization of the scaling-squaring technique and Padé approximation [3]. Therefore, we will focus on how to handle the matrix power function efficiently.

- **Forward process.** The orthogonal matrix $O$ can be factorized, utilizing eigendecomposition, as $O = QXQ^{-1}$, where $Q \in \mathbb{R}^{n \times n}$ with each column representing an eigenvector of $O$, and $X = \mathrm{diag}(\{\lambda_1, \ldots, \lambda_n\})$ is a diagonal matrix whose elements are the eigenvalues of $O$. In this case, the matrix power $O^\tau$ can be computed by applying the power function to the eigenvalues while keeping the eigenvectors unchanged [24], yielding $O^\tau = Q\mathrm{diag}(\{\lambda_1^\tau, \ldots, \lambda_n^\tau\})Q^{-1}$. It is evident that calculating $O^\tau$ is not significantly more complex than computing $n$ scalar powers.

- **Backward process.** Given an orthogonal matrix $O$, we assume that $\widetilde{O} = \psi_\tau(O)$ has been obtained through Equation (11). Then, there exists a unique orthogonal matrix $W_\tau$ such that $\widetilde{O} = W_\tau O$ due to the closure property of the Lie group. Therefore, in the forward pass, one can initially acquire $\widetilde{O}$ using Equation (11), followed by computing the equivalent transformation of the mapping $\psi_\tau$ as $W_\tau = \widetilde{O}O^\top$. In this way, the forward pass is streamlined into $\widetilde{O} = W_\tau O$, thereby rendering the backward pass highly efficient, as it only involves one linear transformation.

### 3.3 Re-parameterization provides stochastic optimization

In the previous section, we considered deterministic optimization over permutation matrices. However, in many scenarios, we commonly build a probabilistic model to express the uncertainty inherent in the problem [7]. For such a task, it is crucial to have the ability to learn latent variable models associated with the latent nodes corresponding to permutations [20]. This section demonstrates how to perform stochastic optimization over the latent permutations using OT4P and the re-parameterization trick.

We restrict our attention to the scenario where the latent variable is a permutation matrix, $z = P$, without loss of generality. Therefore, consider the probabilistic form of Equation (1) as follows

$$\min \mathbb{E}_{P \sim q(P;\theta)} f(P). \tag{14}$$

The above equation deals with a distribution over permutation matrices rather than a single permutation matrix as in Equation (1). Evaluating and differentiating Equation (14) is challenging due to the expectation involving a sum of $n!$ terms. To remedy this, we employ the re-parameterization trick [32, 56, 14]. In particular, we simulate $q(P;\theta)$ using the mappings $\rho(\cdot)$ w.r.t. Equation (7) and $\phi(\cdot)$ w.r.t. Equation (11), expressed as

$$P \sim q(P;\theta) \Longleftrightarrow P = \rho(\phi(A + B\epsilon)) \text{ with } \theta := \{A, B \in \mathbb{R}^{\frac{n(n-1)}{2}}\}, \tag{15}$$

where $\epsilon \sim q(\epsilon)$ is a random noise distribution. Equation (15) cleverly decouples the stochastic nature of the distribution $q(P;\theta)$, thereby obviating the dependence of the expectation on the parameters $\theta$. This enables us to draw multiple samples for the evaluation of Equation (14) with lower variance.

However, there exists a not differentiable mapping $\rho(\cdot)$, which hinders gradient-based optimization for $\theta$. Recalling OT4P, we can approximate Equation (15) by relaxing the mapping $\rho(\cdot)$ to $\psi_\tau(\cdot)$. It is evident that samples drawn from distribution $\psi_\tau(\phi(A + B\epsilon))$ converge almost surely to those from distribution $\rho(\phi(A + B\epsilon))$ owing to $\lim_{\tau \to 0} \psi_\tau = \rho$. To summarize, we can bring the gradient inside the expectation, as depicted below:

$$\nabla \mathbb{E}_{\epsilon \sim q(\epsilon)} f\left(\psi_\tau(\phi(A + B\epsilon))\right) = \mathbb{E}_{\epsilon \sim q(\epsilon)} \nabla f\left(\psi_\tau(\phi(A + B\epsilon))\right), \tag{16}$$

which can now be computed using Monte Carlo [50].

## 4  Experiments

This section conducts experiments to evaluate the performance of OT4P in optimization problems and probabilistic tasks. All experimental details not stated here, along with additional results, can be found in Appendix F. The core code for OT4P is available at https://github.com/YamingGuo98/OT4P.

### 4.1  Finding mode connectivity

In the first experiment, we consider an optimization problem inspired by the concept of linear mode connectivity. Recent studies have shown that neural networks trained with SGD belong to a set whose weights can be permuted so that we linearly connect those weights with no detriment to the loss [13, 57]. For demonstration purposes, we examine a multi-layer perceptron (MLP) with $L$ layers and denote its weights as $\theta = \{W_l | l \in [L]\}$. To find the optimal permutation between models $\theta_A$ and $\theta_B$, Ainsworth et al. [2] propose the following data-free optimization problem:

$$\min_{\pi = \{P_i \in \mathcal{P}_{n_i}\}} \|W_1^{(A)} - P_1 W_1^{(B)}\|_{\mathrm{F}}^2 + \|W_2^{(A)} - P_2 W_2^{(B)} P_1^\top\|_{\mathrm{F}}^2 + \cdots + \|W_L^{(A)} - P_L W_L^{(B)}\|_{\mathrm{F}}^2. \tag{17}$$

The above problem is challenging because it does not admit a polynomial-time constant-factor approximation scheme [2]. We can use OT4P to relax permutation matrices, as demonstrated in Section 3.2, enabling a gradient-based solution to Equation (17).

We explore a variety of network architectures, including MLP5 (5-layer MLP) [54], VGG11 [60], and ResNet18 [22]. The weights for these networks are derived from official pre-trained models in PyTorch [52], with the exception of the MLP5, which is initialized randomly. For model $\theta_B$, we randomly sample permutation matrices from a uniform distribution and apply them to permute the weights, yielding model $\theta_A = \pi(\theta_B)$. The AdamW [45] with an initial learning rate of 0.1 is employed to minimize the loss w.r.t. Equation (17), with a maximum of 500 iterations. To evaluate the results, we use the $\ell_1$-Distance, $\|\theta_A - \pi(\theta_B)\|_1$, to measure the difference from the target weights. Additionally, we flatten the permutation matrices and evaluate their alignment with the ground truth using Precision, Recall, and Hamming Distance.

We compare: 1) Weight Matching [2], which goes through each layer and greedily selects its best permutation matrix $P_i$; 2) Sinkhorn [54], relaxing the permutation matrices to the vicinity of the Birkhoff polytope utilizing the Sinkhorn operator [61, 49]; and 3) OT4P, our proposed method, which is evaluated with various temperature parameters. The results are reported in Tables 1 and 4, with each experiment conducted five times. The findings indicate that Weight Matching occasionally fails to reach ground truth due to its sensitivity to random initialization. Additionally, Sinkhorn yields poor results in the VGG11 network architecture, which we attribute to the relaxation on the Birkhoff polytope producing unreliable local minima. In contrast, OT4P finds the optimal permutation matrix in most cases. Indeed, a neural network can be conceptualized as a geometric object whose vertices correspond to the rows of the weight matrices [35]. A reasonable relaxation of the matching task in Equation (17) involves rigid transformations represented by orthogonal matrices. The proposed OT4P relaxes the permutation matrices into the orthogonal group, which is likely the primary reason for its powerful performance.

### 4.2  Inferring neuron identities

In the second experiment, we tackle a probabilistic task motivated by the study of the neural dynamics in C. elegans [66]. This worm serves as a model organism in neuroscience, with its complete neuronal connectivity known and represented by the adjacency matrix $A \in \{0, 1\}^{n \times n}$. However, matching

Table 1: $\ell_1$-Distance (converted by $\log(1+x)$) and Precision (%) of algorithms for finding mode connectivity across different network architectures.

| | MLP5 | | VGG11 | | ResNet18 | |
|---|---|---|---|---|---|---|
| Algorithm | $\log(1+\ell_1)$ ($\downarrow$) | Precision ($\uparrow$) | $\log(1+\ell_1)$ ($\downarrow$) | Precision ($\uparrow$) | $\log(1+\ell_1)$ ($\downarrow$) | Precision ($\uparrow$) |
| Weight Matching | 0.000 ±0.00 | 100.0 ±0.00 | 0.000 ±0.00 | 100.0 ±0.00 | 1.215 ±2.72 | 99.97 ±0.06 |
| Sinkhorn | 0.000 ±0.00 | 100.0 ±0.00 | 11.61 ±0.07 | 63.08 ±3.14 | 9.830 ±0.181 | 95.56 ±0.88 |
| OT4P ($\tau = 0.3$) | 0.000 ±0.00 | 100.0 ±0.00 | 0.000 ±0.00 | 100.0 ±0.00 | 0.000 ±0.00 | 100.0 ±0.00 |
| OT4P ($\tau = 0.5$) | 0.000 ±0.00 | 100.0 ±0.00 | 0.818 ±1.83 | 99.99 ±0.03 | 0.000 ±0.00 | 100.0 ±0.00 |
| OT4P ($\tau = 0.7$) | 0.000 ±0.00 | 100.0 ±0.00 | 0.000 ±0.00 | 100.0 ±0.00 | 0.000 ±0.00 | 100.0 ±0.00 |

traces from the observed neural dynamics $Y \in \mathbb{R}^{n \times 1}$ to the neurons in the reference connectome $A$ poses a challenging task. Linderman et al. [43] propose simulating neural activity using a linear dynamical system $Y_t = P(A \odot W)P^\top Y_{t-1} + \epsilon$, where $\epsilon$ is Gaussian noise, $W \in \mathbb{R}^{n \times n}, P \in \mathcal{P}_n$ are latent variables, and $\odot$ is element-wise product. Our goal is to infer the latent permutation $P$ to align the observed $Y$ with the shared dynamics matrix $W$. We address this task by maximizing the marginal log-likelihood, i.e., $\max \mathbb{E}_{P \sim q(P;\theta)} \log p(Y|P)$, using the techniques outlined in Section 3.3.

Taking the methodology in Linderman et al. [43], we generate parameters $A$, $W$, and $P$ with $n = 250$ and randomly generate 1000 samples, where the noise follows a Gaussian distribution $\mathcal{N}(0, 0.01)$. We formulate tasks of varying difficulty depending on the different proportions of known neurons [43]. Conceiving a constraint matrix $C \in \mathbb{R}^{n \times n}$ where all elements are initialized to 1, if we ascertain that the reference neuron $i$ corresponds to the observed neuron $j$, then set all elements to 0 in the $i$-th row and $j$-th column except for $C_{i,j}$ (see Equation (22) for an example). This constraint is enforced by zeroing corresponding entries before solving Equation (7). We conduct 500 iterations using the Adam optimizer [31] with an initial learning rate of 0.01. We report the marginal log-likelihood of the best model throughout the training, ranked first by Hamming Distance and then by the marginal log-likelihood (estimated with 5 repeats). As done in Section 4.1, Precision, Recall, and Hamming Distance are utilized to evaluate the permutation matrices obtained from the best model.

For comparison, we include: 1) `Naive` [43], which does not enforce that $P$ is a permutation matrix and instead normalizes each row using the softmax function; 2) `Gumbel-Sinkhorn` [49], introducing Gumbel noise for re-parameterization before the Sinkhorn operator; and 3) `OT4P`, our proposed method using the re-parameterization trick, with different temperature parameters. Each experiment is conducted five times, and the results are presented in Tables 2 and 5. We observe that `Naive` fails to produce any meaningful solutions, and `Gumbel-Sinkhorn` performs poorly in the more challenging scenario (Known 5%). In contrast, `OT4P` consistently identifies the optimal permutation, except for `OT4P` ($\tau = 0.7$), which achieves suboptimal results in the Known 5% setting. One possible reason why `OT4P` performs better is that the orthogonal group ($\frac{n(n-1)}{2}$) has a lower dimension than the Birkhoff polytope ($(n-1)^2$). This lower dimensionality makes the randomness simulated by the noise more effective in exploring latent permutations.

Table 2: Marginal log-likelihood and Precision (%) of algorithms for inferring neuron identities across different proportions of known neurons.

| | Known 5% | | Known 10% | | Known 20% | |
|---|---|---|---|---|---|---|
| Algorithm | $\mathbb{E} \log p(Y|P)$ ($\uparrow$) | Precision ($\uparrow$) | $\mathbb{E} \log p(Y|P)$ ($\uparrow$) | Precision ($\uparrow$) | $\mathbb{E} \log p(Y|P)$ ($\uparrow$) | Precision ($\uparrow$) |
| Naive | −3040 ±43.4 | 8.960 ±7.85 | −2917 ±225 | 29.68 ±17.2 | −1690 ±539 | 78.40 ±12.6 |
| Gumbel-Sinkhorn | −2256 ±574 | 62.08 ±16.0 | −239.8 ±119 | 98.16 ±1.95 | −144.8 ±27.1 | 99.84 ±0.358 |
| OT4P ($\tau = 0.3$) | −130.9 ±10.9 | 100.0 ±0.00 | −127.5 ±10.1 | 100.0 ±0.00 | −126.7 ±11.0 | 100.0 ±0.00 |
| OT4P ($\tau = 0.5$) | −164.0 ±36.8 | 100.0 ±0.00 | −149.7 ±25.0 | 100.0 ±0.00 | −148.2 ±27.6 | 100.0 ±0.00 |
| OT4P ($\tau = 0.7$) | −829.3 ±831 | 74.16 ±35.9 | −183.1 ±46.2 | 100.0 ±0.00 | −171.8 ±40.3 | 100.0 ±0.00 |

## 4.3 Solving permutation synchronization

In the third experiment, we aim to explore the effectiveness of our proposed `OT4P` on large-scale problems. We specifically focus on the permutation synchronization problem [51, 47, 5], which tries to improve matching across multiple objects. Consider $k$ objects with $n$ points each, and let the permutation matrix $P(i,j) \in \mathcal{P}_n$ to represent the correspondence between points in objects $i$ and

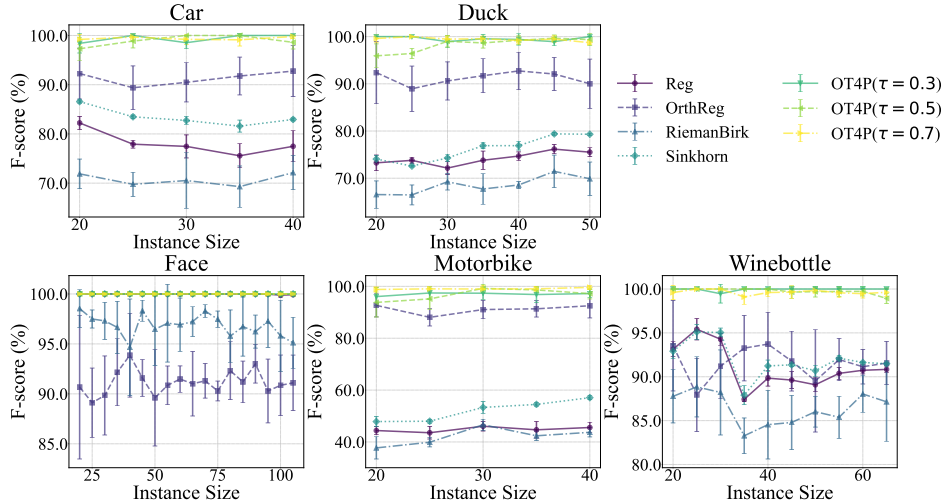

Figure 3: F-scores (%) for different algorithms on the WILLOW-ObjectClass dataset, where the size of permutation synchronization problem instances varies along the horizontal axis.

$j$. Permutation synchronization seeks to identify the underlying permutations $P_i (i \in [k])$ such that $P(i,j) = P_i P_j^\top$ for all $i, j \in [k]$, which can be expressed as the following optimization problem:

$$\min_{\{P_i \in \mathcal{P}_n\}} \sum_{i,j}^{k} \|P(i,j) - P_i P_j^\top\|_{\mathrm{F}}^2. \tag{18}$$

We select baselines: 1) `Reg` optimizes in Euclidean space with a regularization term $\sum_j (\sum_j P_{i,j} - 1)^2$ that encourages each column to sum to 1. 2) `OrthReg` [72] optimizes over the special orthogonal group, using a regularization term $\frac{2}{3} \mathrm{trace}(P^T(P - P \odot P))$ ($\odot$ is element-wise product) to force orthogonal matrices to converge to permutation matrices. 3) `RiemanBirk` [6] optimizes on Birkhoff polytope utilizing Riemannian gradient descent. 4) `Sinkhorn` [49] optimizes in the vicinity of the Birkhoff polytope, using the Sinkhorn operator to adjust positive matrices into approximate doubly stochastic matrices. All algorithms employ the Adam optimizer for 100 iterations, with `RiemanBirk` utilizing Riemannian Adam [4, 34]. The initial learning rates are tuned within the set $\{0.1, 0.01, 0.001, 0.0001\}$.

We use the WILLOW-ObjectClass dataset [9] to generate problem instances (see Appendix F.4 for more details) and utilize the F-score to evaluate the alignment between the flattened permutation matrices and the ground truth. Each experiment is conducted five times, and the results are shown in Figure 3. `RiemanBirk` and `Sinkhorn` demonstrate poorer performance. A primary reason is that both methods are based on Birkhoff polytope to relax permutations, leading to unreliable local minima and preventing optimal solutions. Benefiting from the potential advantages offered by the orthogonal group, `OrthReg` generally produces competitive results. However, due to the instability of its regularization term, `OrthReg` sometimes underperforms, which may necessitate careful adjustment of the regularization coefficient for each class. In contrast, our proposed `OT4P` consistently outperforms other methods and demonstrates robustness to variations in the hyperparameter $\tau$.

## 5 Conclusion

In this paper, we present a novel differentiable transformation, `OT4P`, designed for relaxing permutation matrices over the orthogonal group. This method is characterized by its flexibility, simplicity, and scalability. `OT4P` is utilized to parametrize the relaxation of permutation matrices, with advantages of not altering the original problem, not complicating the original problem, and an efficient optimization process. By deriving a gradient estimator, `OT4P` further provides an efficient tool for stochastic optimization over latent permutations. Extensive experiments show that `OT4P` achieves competitive results in optimization problems and probabilistic tasks compared to relaxation methods on the Birkhoff polytope. We believe our elementary work is a significant step toward relaxing permutations onto the orthogonal group. Please see Appendix A for further discussion, including related work, limitations, and broader impacts.

## Acknowledgments and Disclosure of Funding

This work is supported by the National Key Research and Development Program of China (No. 2020YFA0714103).

## Footnotes

[2] Here, the term 'logm' refers to the generalized matrix logarithm, accommodating instances where the principal logarithm may not be defined.

## References

[1] Ryan Prescott Adams and Richard S Zemel. Ranking via sinkhorn propagation. *arXiv preprint arXiv:1106.1925*, 2011.

[2] Samuel Ainsworth, Jonathan Hayase, and Siddhartha Srinivasa. Git re-basin: Merging models modulo permutation symmetries. In *The Eleventh International Conference on Learning Representations*, 2023. URL https://openreview.net/forum?id=CQsmMYmlP5T.

[3] Awad H Al-Mohy and Nicholas J Higham. A new scaling and squaring algorithm for the matrix exponential. *SIAM Journal on Matrix Analysis and Applications*, 31(3):970–989, 2010.

[4] Gary Becigneul and Octavian-Eugen Ganea. Riemannian adaptive optimization methods. In *International Conference on Learning Representations*, 2019. URL https://openreview.net/forum?id=r1eiqi09K7.

[5] Florian Bernard, Daniel Cremers, and Johan Thunberg. Sparse quadratic optimisation over the stiefel manifold with application to permutation synchronisation. *Advances in Neural Information Processing Systems*, 34:25256–25266, 2021.

[6] Tolga Birdal and Umut Simsekli. Probabilistic permutation synchronization using the riemannian structure of the birkhoff polytope. In *Proceedings of the IEEE/CVF Conference on Computer Vision and Pattern Recognition*, pages 11105–11116, 2019.

[7] David M Blei, Alp Kucukelbir, and Jon D McAuliffe. Variational inference: A review for statisticians. *Journal of the American statistical Association*, 112(518):859–877, 2017.

[8] Rainer E Burkard, Stefan E Karisch, and Franz Rendl. Qaplib–a quadratic assignment problem library. *Journal of Global optimization*, 10:391–403, 1997.

[9] Minsu Cho, Karteek Alahari, and Jean Ponce. Learning graphs to match. In *Proceedings of the IEEE International Conference on Computer Vision*, pages 25–32, 2013.

[10] Jia Deng, Wei Dong, Richard Socher, Li-Jia Li, Kai Li, and Li Fei-Fei. Imagenet: A large-scale hierarchical image database. In *2009 IEEE conference on computer vision and pattern recognition*, pages 248–255. Ieee, 2009.

[11] Zvi Drezner. The quadratic assignment problem. *Location science*, pages 345–363, 2015.

[12] Hannah Dröge, Zorah Lähner, Yuval Bahat, Onofre Martorell Nadal, Felix Heide, and Michael Möller. Kissing to find a match: Efficient low-rank permutation representation. *Advances in Neural Information Processing Systems*, 36, 2024.

[13] Rahim Entezari, Hanie Sedghi, Olga Saukh, and Behnam Neyshabur. The role of permutation invariance in linear mode connectivity of neural networks. In *International Conference on Learning Representations*, 2022. URL https://openreview.net/forum?id=dNigytemkL.

[14] Luca Falorsi, Pim de Haan, Tim R Davidson, and Patrick Forré. Reparameterizing distributions on lie groups. In *The 22nd International Conference on Artificial Intelligence and Statistics*, pages 3244–3253. PMLR, 2019.

[15] Marcelo Fiori, Pablo Sprechmann, Joshua Vogelstein, Pablo Musé, and Guillermo Sapiro. Robust multimodal graph matching: Sparse coding meets graph matching. *Advances in neural information processing systems*, 26, 2013.

[16] Matteo Fischetti, Michele Monaci, and Domenico Salvagnin. Three ideas for the quadratic assignment problem. *Operations research*, 60(4):954–964, 2012.

[17] Fajwel Fogel, Rodolphe Jenatton, Francis Bach, and Alexandre d'Aspremont. Convex relaxations for permutation problems. *Advances in neural information processing systems*, 26, 2013.

[18] Jean Gallier. Logarithms and square roots of real matrices existence, uniqueness and applications in medical imaging. *arXiv preprint arXiv:0805.0245*, 2011.

[19] Zheng Gong and Ying Sun. An energy-centric framework for category-free out-of-distribution node detection in graphs. In *Proceedings of the 30th ACM SIGKDD Conference on Knowledge Discovery and Data Mining*, pages 908–919, 2024.

[20] Aditya Grover, Eric Wang, Aaron Zweig, and Stefano Ermon. Stochastic optimization of sorting networks via continuous relaxations. In *International Conference on Learning Representations*, 2019. URL https://openreview.net/forum?id=H1eSS3CcKX.

[21] Brian C Hall. *Lie groups, Lie algebras, and representations*. Springer, 2013.

[22] Kaiming He, Xiangyu Zhang, Shaoqing Ren, and Jian Sun. Deep residual learning for image recognition. In *Proceedings of the IEEE conference on computer vision and pattern recognition*, pages 770–778, 2016.

[23] Nicholas J Higham. *Functions of matrices: theory and computation*. SIAM, 2008.

[24] Nicholas J Higham and Lijing Lin. A schur–padé algorithm for fractional powers of a matrix. *SIAM Journal on Matrix Analysis and Applications*, 32(3):1056–1078, 2011.

[25] Einar Hille. On roots and logarithms of elements of a complex banach algebra. *Mathematische Annalen*, 136(1):46–57, 1958.

[26] Bo Jiang, Ya-Feng Liu, and Zaiwen Wen. L_p-norm regularization algorithms for optimization over permutation matrices. *SIAM Journal on Optimization*, 26(4):2284–2313, 2016.

[27] Bo Jiang, Jin Tang, Chris Ding, and Bin Luo. Nonnegative orthogonal graph matching. In *Proceedings of the AAAI Conference on Artificial Intelligence*, volume 31, 2017.

[28] Ivan Karpukhin and Andrey Savchenko. Detpp: Leveraging object detection for robust long-horizon event prediction. *arXiv preprint arXiv:2408.13131*, 2024.

[29] Ivan Karpukhin, Foma Shipilov, and Andrey Savchenko. Hotpp benchmark: Are we good at the long horizon events forecasting? *arXiv preprint arXiv:2406.14341*, 2024.

[30] Sunyoung Kim, Masakazu Kojima, and Kim-Chuan Toh. A lagrangian–dnn relaxation: a fast method for computing tight lower bounds for a class of quadratic optimization problems. *Mathematical Programming*, 156(1):161–187, 2016.

[31] Diederik P Kingma and Jimmy Ba. Adam: A method for stochastic optimization. *arXiv preprint arXiv:1412.6980*, 2014.

[32] Diederik P Kingma and Max Welling. Auto-encoding variational bayes. *arXiv preprint arXiv:1312.6114*, 2013.

[33] Virginia Klema and Alan Laub. The singular value decomposition: Its computation and some applications. *IEEE Transactions on automatic control*, 25(2):164–176, 1980.

[34] Max Kochurov, Rasul Karimov, and Serge Kozlukov. Geoopt: Riemannian optimization in pytorch, 2020.

[35] Vignesh Kothapalli. Neural collapse: A review on modelling principles and generalization. *arXiv preprint arXiv:2206.04041*, 2022.

[36] Harold W Kuhn. The hungarian method for the assignment problem. *Naval research logistics quarterly*, 2(1-2):83–97, 1955.

[37] Eugene L Lawler. The quadratic assignment problem. *Management science*, 9(4):586–599, 1963.

[38] John M Lee. *Introduction to Riemannian manifolds*, volume 2. Springer, 2018.

[39] Mario Lezcano Casado. Trivializations for gradient-based optimization on manifolds. *Advances in Neural Information Processing Systems*, 32, 2019.

[40] Mario Lezcano-Casado and David Martınez-Rubio. Cheap orthogonal constraints in neural networks: A simple parametrization of the orthogonal and unitary group. In *International Conference on Machine Learning*, pages 3794–3803. PMLR, 2019.

[41] Innar Liiv. Seriation and matrix reordering methods: An historical overview. *Statistical Analysis and Data Mining: The ASA Data Science Journal*, 3(2):70–91, 2010.

[42] Cong Han Lim and Stephen Wright. Beyond the birkhoff polytope: Convex relaxations for vector permutation problems. *Advances in neural information processing systems*, 27, 2014.

[43] Scott Linderman, Gonzalo Mena, Hal Cooper, Liam Paninski, and John Cunningham. Reparameterizing the birkhoff polytope for variational permutation inference. In *International Conference on Artificial Intelligence and Statistics*, pages 1618–1627. PMLR, 2018.

[44] Eliane Maria Loiola, Nair Maria Maia De Abreu, Paulo Oswaldo Boaventura-Netto, Peter Hahn, and Tania Querido. A survey for the quadratic assignment problem. *European journal of operational research*, 176(2):657–690, 2007.

[45] Ilya Loshchilov and Frank Hutter. Decoupled weight decay regularization. *arXiv preprint arXiv:1711.05101*, 2017.

[46] Chris J Maddison, Andriy Mnih, and Yee Whye Teh. The concrete distribution: A continuous relaxation of discrete random variables. *arXiv preprint arXiv:1611.00712*, 2016.

[47] Eleonora Maset, Federica Arrigoni, and Andrea Fusiello. Practical and efficient multi-view matching. In *Proceedings of the IEEE International Conference on Computer Vision*, pages 4568–4576, 2017.

[48] Estelle Massart and Vinayak Abrol. Coordinate descent on the orthogonal group for recurrent neural network training. In *Proceedings of the AAAI Conference on Artificial Intelligence*, volume 36, pages 7744–7751, 2022.

[49] Gonzalo Mena, David Belanger, Scott Linderman, and Jasper Snoek. Learning latent permutations with gumbel-sinkhorn networks. In *International Conference on Learning Representations*, 2018. URL https://openreview.net/forum?id=Byt3oJ-0W.

[50] Shakir Mohamed, Mihaela Rosca, Michael Figurnov, and Andriy Mnih. Monte carlo gradient estimation in machine learning. *Journal of Machine Learning Research*, 21(132):1–62, 2020.

[51] Deepti Pachauri, Risi Kondor, and Vikas Singh. Solving the multi-way matching problem by permutation synchronization. *Advances in neural information processing systems*, 26, 2013.

[52] Adam Paszke, Sam Gross, Francisco Massa, Adam Lerer, James Bradbury, Gregory Chanan, Trevor Killeen, Zeming Lin, Natalia Gimelshein, Luca Antiga, et al. Pytorch: An imperative style, high-performance deep learning library. *Advances in neural information processing systems*, 32, 2019.

[53] Gao Peifeng, Qianqian Xu, Peisong Wen, Zhiyong Yang, Huiyang Shao, and Qingming Huang. Feature directions matter: Long-tailed learning via rotated balanced representation. In *International Conference on Machine Learning*, pages 27542–27563. PMLR, 2023.

[54] Fidel A Guerrero Peña, Heitor Rapela Medeiros, Thomas Dubail, Masih Aminbeidokhti, Eric Granger, and Marco Pedersoli. Re-basin via implicit sinkhorn differentiation. In *Proceedings of the IEEE/CVF Conference on Computer Vision and Pattern Recognition*, pages 20237–20246, 2023.

[55] Janez Povh. Semidefinite approximations for quadratic programs over orthogonal matrices. *Journal of Global Optimization*, 48(3):447–463, 2010.

[56] Danilo Jimenez Rezende, Shakir Mohamed, and Daan Wierstra. Stochastic backpropagation and approximate inference in deep generative models. In *International conference on machine learning*, pages 1278–1286. PMLR, 2014.

[57] Simone Rossi, Ankit Singh, and Thomas Hannagan. On permutation symmetries in bayesian neural network posteriors: a variational perspective. *Advances in Neural Information Processing Systems*, 36, 2024.

[58] Sebastian Ruder. An overview of gradient descent optimization algorithms. *arXiv preprint arXiv:1609.04747*, 2016.

[59] Rodrigo Santa Cruz, Basura Fernando, Anoop Cherian, and Stephen Gould. Deeppermnet: Visual permutation learning. In *Proceedings of the IEEE Conference on Computer Vision and Pattern Recognition*, pages 3949–3957, 2017.

[60] Karen Simonyan and Andrew Zisserman. Very deep convolutional networks for large-scale image recognition. *arXiv preprint arXiv:1409.1556*, 2014.

[61] Richard Sinkhorn. A relationship between arbitrary positive matrices and doubly stochastic matrices. *The annals of mathematical statistics*, 35(2):876–879, 1964.

[62] Ying Sun, Hengshu Zhu, Chuan Qin, Fuzhen Zhuang, Qing He, and Hui Xiong. Discerning decision-making process of deep neural networks with hierarchical voting transformation. *Advances in Neural Information Processing Systems*, 34:17221–17234, 2021.

[63] Ying Sun, Fuzhen Zhuang, Hengshu Zhu, Qing He, and Hui Xiong. Cost-effective and interpretable job skill recommendation with deep reinforcement learning. In *Proceedings of the Web Conference 2021*, pages 3827–3838, 2021.

[64] Ying Sun, Hengshu Zhu, Lu Wang, Le Zhang, and Hui Xiong. Large-scale online job search behaviors reveal labor market shifts amid covid-19. *Nature Cities*, 1(2):150–163, 2024.

[65] David M Tate and Alice E Smith. A genetic approach to the quadratic assignment problem. *Computers & Operations Research*, 22(1):73–83, 1995.

[66] Lav R Varshney, Beth L Chen, Eric Paniagua, David H Hall, and Dmitri B Chklovskii. Structural properties of the caenorhabditis elegans neuronal network. *PLoS computational biology*, 7(2): e1001066, 2011.

[67] Chao Wang, Hengshu Zhu, Chen Zhu, Chuan Qin, and Hui Xiong. Setrank: A setwise bayesian approach for collaborative ranking from implicit feedback. In *Proceedings of the aaai conference on artificial intelligence*, volume 34, pages 6127–6136, 2020.

[68] Chao Wang, Hengshu Zhu, Chen Zhu, Chuan Qin, Enhong Chen, and Hui Xiong. Setrank: A setwise bayesian approach for collaborative ranking in recommender system. *ACM Transactions on Information Systems*, 42(2):1–32, 2023.

[69] Runzhong Wang, Junchi Yan, and Xiaokang Yang. Neural graph matching network: Learning lawler's quadratic assignment problem with extension to hypergraph and multiple-graph matching. *IEEE Transactions on Pattern Analysis and Machine Intelligence*, 44(9):5261–5279, 2021.

[70] Tianxin Wang, Fuzhen Zhuang, Ying Sun, Xiangliang Zhang, Leyu Lin, Feng Xia, Lei He, and Qing He. Adaptively sharing multi-levels of distributed representations in multi-task learning. *Information Sciences*, 591:226–234, 2022.

[71] Junchi Yan, Xu-Cheng Yin, Weiyao Lin, Cheng Deng, Hongyuan Zha, and Xiaokang Yang. A short survey of recent advances in graph matching. In *Proceedings of the 2016 ACM on international conference on multimedia retrieval*, pages 167–174, 2016.

[72] Michael M Zavlanos and George J Pappas. A dynamical systems approach to weighted graph matching. *Automatica*, 44(11):2817–2824, 2008.

[73] He Zhang, Ying Sun, Weiyu Guo, Yafei Liu, Haonan Lu, Xiaodong Lin, and Hui Xiong. Interactive interior design recommendation via coarse-to-fine multimodal reinforcement learning. In *Proceedings of the 31st ACM International Conference on Multimedia*, pages 6472–6480, 2023.

[74] Yuting Zhang, Ying Sun, Fuzhen Zhuang, Yongchun Zhu, Zhulin An, and Yongjun Xu. Triple dual learning for opinion-based explainable recommendation. *ACM Transactions on Information Systems*, 42(3):1–27, 2023.

[75] Hengshu Zhu, Hui Xiong, Yong Ge, and Enhong Chen. Discovery of ranking fraud for mobile apps. *IEEE Transactions on knowledge and data engineering*, 27(1):74–87, 2014.

# A Additional discussions

## A.1 Related work

Optimization over permutations is commonly encountered in problems involving bijections between two sets of equal size [17, 42, 15, 12]. From a theoretical perspective, a well-known special case is the quadratic assignment problem, which has been drawing researchers' attention since its first formulation [37, 8, 44]. In practical terms, optimization over permutations is also arises extensively in various machine learning tasks, including ranking [20, 75, 67, 68], matching [2], tracking [43], recommendation [63, 74, 73], etc. Since our interest lies in addressing generic problems defined on permutation matrices, we will not discuss theoretical approaches for specific issues [11], such as exact methods [16], lower bound methods [30], and vertex-based methods [65].

Previous studies have proposed relaxing the permutation matrices to continuous spaces, including the Birkhoff polytope [17, 15, 42] and orthogonal group [72, 55, 27]. The Birkhoff polytope is the convex hull of all permutation matrices, while the orthogonal group serves as a natural embedding of permutation matrices in a differentiable manifold. Such relaxation techniques have proven to be very powerful, with successful applications to various problems such as seriation [1, 17, 41] and graph matching [15, 69, 71]. Most of the methods mentioned above rely on the explicit penalty term and are challenging to extend to probabilistic scenarios.

Recently, relaxation methods involving the Birkhoff polytope have made significant advancements, particularly in penalty-free optimization and probabilistic inference. For instance, Linderman et al. [43] propose a rounding transformation regulated by a temperature parameter, which rounds matrices towards the vertices of the Birkhoff polytope, i.e., permutation matrices. Similarly, Mena et al. [49] utilize the Sinkhorn operator [61] to transform matrices into the Birkhoff polytope, bringing them closer to permutation matrices under temperature control. Additionally, Grover et al. [20] suggest mapping vectors to unimodal row stochastic matrices, a subset of the Birkhoff polytope that removes the requirement of every column sum being equal to 1. These works not only avoid introducing penalty terms but also provide probabilistic inference. However, providing equally effective relaxation methods within orthogonal groups remains an unexplored area. As mentioned in the main text, relaxation onto the orthogonal group possesses unique potential advantages. Therefore, our elementary work aims to address this gap.

## A.2 Limitations

The first concerns the computational efficiency of OT4P when dealing with very large matrices ($n > 1000$), as the cost of eigendecomposition becomes prohibitive. This could benefit from efficient implementations of eigendecomposition on GPUs. The second limitation involves the noise distribution used for re-parameterization in OT4P, which may not equally capture the latent permutation matrices. This can be improved by more carefully designing the noise distribution under the characteristics of the orthogonal group. The third challenge relates to the boundary issues in representing permutation matrices, hindering OT4P as a building block of deep neural networks. Indeed, such integration assumes all permutation matrices lie within $\mathcal{U}$. We believe that an ideal (analytical) solution can be devised to relocate all permutation matrices back into $\mathcal{U}$, given that they constitute a finite discrete set within the orthogonal group of order $n$.

## A.3 Broader impacts

This work presents a novel differentiable transformation for relaxing permutation matrices onto the orthogonal group, which enables gradient-based (stochastic) optimization of problems involving permutation matrices. Given the theoretical nature of our work, we have not identified any direct ethical concerns or negative societal impacts related to our research. Our study may have the broader impacts for a variety of areas of machine learning, such as deep learning [62], data mining [64]. Similar impacts were observed in multi-task learning [70] and graph learning [19].

# B Riemannian Geometry and Lie Group

This section briefly summarizes the key concepts of Riemannian geometry and the Lie group theory involved. For a comprehensive understanding, we recommend the standard textbooks *Introduction to*

*Riemannian manifolds* [38] and *Lie groups, Lie algebras, and representations* [21]. Additionally, the analysis of matrix functions can be found in *Functions of matrices: theory and computation* [23].

An $n$-dimensional *manifold* $\mathcal{M}$ is a topological space that can be locally approximated by the Euclidean space $\mathbb{R}^n$. At each point $x \in \mathcal{M}$, there exists a *tangent space* $T_x\mathcal{M}$, an $n$-dimensional vector space, serving as a first-order local approximation of $\mathcal{M}$ around $x$. The *Riemannian metric* is a collection $m := \{m_x \mid x \in \mathcal{M}\}$ of inner products $m_x(\cdot, \cdot) : T_x\mathcal{M} \times T_x\mathcal{M} \to \mathbb{R}$. It induces a *norm* $\|\cdot\|_x : T_x\mathcal{M} \to \mathbb{R}$ defined by $\|y\|_x := \sqrt{m_x(y, y)}$. The *length* $L(\gamma)$ of a smooth curve $\gamma : [a, b] \to \mathcal{M}$ is defined as $L(\gamma) := \int_a^b \|\gamma'(t)\|_{\gamma(t)} \, \mathrm{d}t$. The *distance* $d(x, y)$ is set as the infimum of the lengths of all smooth curves between $x$ and $y$ in $\mathcal{M}$. A *geodesic* is a smooth curve $\gamma : [a, b] \to \mathcal{M}$ that the tangent vector $\gamma'(t)$ is parallel transported along the curve $\gamma$ w.r.t. the Levi-Civita connection, i.e., $\nabla_{\gamma'(t)}\gamma'(t) = 0$ for all $t \in [a, b]$. With the basic concept of manifolds established, we can present the definitions of isomorphism, diffeomorphism, and isometry.

**Definition 1** (Vector space isomorphism). The vector spaces $\mathcal{X}$ and $\mathcal{Y}$ are called to be isomorphic if there exists a bijection $\alpha : \mathcal{X} \to \mathcal{Y}$ that preserves addition and scalar multiplication.

**Definition 2** (Diffeomorphism). Given two differentiable manifolds $\mathcal{M}$ and $\mathcal{N}$, a differentiable map $\alpha : \mathcal{M} \to \mathcal{N}$ is a diffeomorphism if it is a bijection and its inverse $\alpha^{-1}$ is differentiable as well.

**Definition 3** (Isometry). Given two metric spaces $(\mathcal{X}, m_{\mathcal{X}})$ and $(\mathcal{Y}, m_{\mathcal{Y}})$, a map $\alpha : \mathcal{X} \to \mathcal{Y}$ is called an isometry if $m_{\mathcal{X}}(x_1, x_2) = m_{\mathcal{Y}}(\alpha(x_1), \alpha(x_2))$ for all points $x_1, x_2 \in \mathcal{X}$.

A *Lie group* $G$ is a differentiable manifold equipped with differentiable group multiplication and inverse operations. The tangent space at the identity $e$ is known as the *Lie algebra* of $G$, denoted as $\mathfrak{g} := T_eG$. For all $g \in G$, there exist diffeomorphisms given by the *left translation* $L_g(x) := gx$ ($\forall x \in G$) and the *right translation* $R_g(x) := xg$ ($\forall x \in G$). The left and right translations lead to a vector space isomorphism that relates the tangent space to the Lie algebra. For left translation $L_g$, the mapping $(\mathrm{d}L_g)_e := \mathfrak{g} \to T_gG$ maps elements of the Lie algebra into the tangent space $T_gG$, with its inverse mapping $(\mathrm{d}L_{g^{-1}})_g := T_gG \to \mathfrak{g}$. Similarly, for right translation $R_g$, there is mapping $(\mathrm{d}R_g)_e := \mathfrak{g} \to T_gG$ and its inverse $(\mathrm{d}R_{g^{-1}})_g := T_gG \to \mathfrak{g}$. A Riemannian metric $m$ on the Lie group $G$ is said to be *left-invariant* (*right-invariant*) if it renders each left (right) translation an isometry. When adopting a left-invariant (right-invariant) metric on the Lie group $G$, we can associate neighborhoods of the identity $e$ with neighborhoods of any point $g \in G$ using left (right) translation, and vice versa. A metric on the Lie group that is both left and right invariant is termed a *bi-invariant* metric. It is worth noting that compact Lie groups always possess the bi-invariant metric.

The Lie group we are interested in is the orthogonal group $\mathrm{O}(n)$, consisting of all $n \times n$ orthogonal matrices $O$ satisfying $O^\top O = OO^\top = I$. We equip $\mathrm{O}(n)$ with the canonical metric, which is a bi-invariant metric inherited from $\mathbb{R}^{n \times n}$ and defined as $\langle A, B \rangle_{\mathrm{F}} := \mathrm{trace}(A^\top B)$, where $\langle \cdot, \cdot \rangle_{\mathrm{F}}$ is the Frobenius inner product and $\mathrm{trace}(\cdot)$ is the trace of a matrix. The orthogonal group $\mathrm{O}(n)$ is divided into two connected components depending on the value of the determinant, $+1$ or $-1$. The connected component with determinant $+1$ forms a subgroup known as the *special orthogonal group* $\mathrm{SO}(n) := \{O \in \mathbb{R}^{n \times n} \mid O^\top O = I, \det O = +1\}$. The Lie algebra $\mathfrak{so}(n)$ of the Lie group $\mathrm{SO(n)}$ comprises $n \times n$ skew-symmetric matrices, expressed as $\mathfrak{so}(n) := \{A \in \mathbb{R}^{n \times n} \mid A^\top = -A\}$. It is noteworthy that $\mathfrak{so}(n)$ is a vector space with dimension $\frac{n(n-1)}{2}$. The *Lie exponential* $\mathrm{expm}(\cdot)$, coinciding with the *matrix exponential* in the context of the matrix Lie group, exactly maps elements in $\mathfrak{so}(n)$ to elements in $\mathrm{SO}(n)$, defined by

$$\mathrm{expm}(A) := I + \sum_{k=1}^{\infty} \frac{A^k}{k!}.$$

The series in the above equation converges for all matrices $A$.

In general, the matrix logarithm is expected to be the inverse function of the matrix exponential, meaning the logarithm of $A$ is the solution $B$ to the matrix equation $\mathrm{expm}(B) = A$. Since the complex logarithm is a multi-valued function, there may be an infinite number of matrices $B$ that satisfy $\mathrm{expm}(B) = A$. If $A$ has no eigenvalues on $\mathbb{R}_0^- := \{x \in \mathbb{R} \mid x \leq 0\}$, then there exists a unique logarithm called the *principal logarithm*, denoted as $\mathrm{logm}(A)$. All eigenvalues $\lambda$ of $\mathrm{logm}(A)$ satisfy $-\pi < \mathrm{Im}\,\lambda < \pi$. Assuming convergence of the series, the *matrix logarithm* $\mathrm{logm}(\cdot)$ can be

defined by Taylor series expansion as follows:

$$\mathrm{logm}(A) := \sum_{k=1}^{\infty} (-1)^{k+1} \frac{(A-I)^k}{k}.$$

The series in the above equation converges whenever $\|A - I\|_{\mathrm{F}} < 1$, where $\|A\|_{\mathrm{F}} := \sqrt{\langle A, A \rangle_{\mathrm{F}}}$ represents the Frobenius norm induced by the Frobenius inner product.

Define the $\mathcal{V} := \{A \in \mathbb{R}^{n \times n} \mid \mathrm{Im}\, \lambda_k(A) \in (-\pi, \pi), \forall k\}$ as the set of all matrices $A \in \mathbb{R}^{n \times n}$ for which all eigenvalues $\lambda_k$ satisfies $-\pi < \mathrm{Im}\, \lambda_k < \pi$. Let $\mathcal{W} := \{A \in \mathbb{R}^{n \times n} \mid \lambda_k(A) \notin \mathbb{R}_0^-, \forall k\}$ denote the set of all matrices $A \in \mathbb{R}^{n \times n}$ without non-positive real eigenvalues. Then, we can verify that (prove in Appendix G.1):

- $\mathrm{logm}(\mathrm{expm}(A)) = A$, $\forall A \in \mathcal{V}$;
- $\mathrm{expm}(\mathrm{logm}(A)) = A$, $\forall A \in \mathcal{W}$.

This indicates that the matrix logarithm is a locally inverse function of the matrix exponential.

## C   Boundary issues in representation of permutation matrix

In this section, we provide an in-depth analysis of the potential adverse effects on optimization stemming from the direct application of mapping $\phi(\cdot)$ w.r.t. Equation (6) . Subsequently, we show that left-multiplying the result of Equation (6) by an orthogonal matrix, as described in Step I, effectively alleviates this problem.

Since a permutation matrix must be an orthogonal matrix, its eigenvalues $\lambda_k$ lie on the unit circle in the complex plane, i.e., $|\lambda_k| = 1$. Additionally, any permutation can be expressed as the product of cycles with disjoint supports. For a cycle of length $m$, its corresponding $m \times m$ submatrix satisfies $P^m = I$, which implies that the eigenvalues of $P$ admit the following form:

$$\lambda_k = \mathrm{e}^{2\pi i \frac{k}{m}}, \quad k = 0, 1, \ldots, m-1. \tag{19}$$

In Figure 4, we visualize the eigenvalues corresponding to cycles of lengths 2, 3, 4, and 5. It is clear that cycles of even length consistently possess an eigenvalue of $-1$. Notably, the eigenvalues of a permutation matrix are composed of those from the submatrices corresponding to its contained cycles. Therefore, many permutation matrices have $-1$ as one of their eigenvalues.

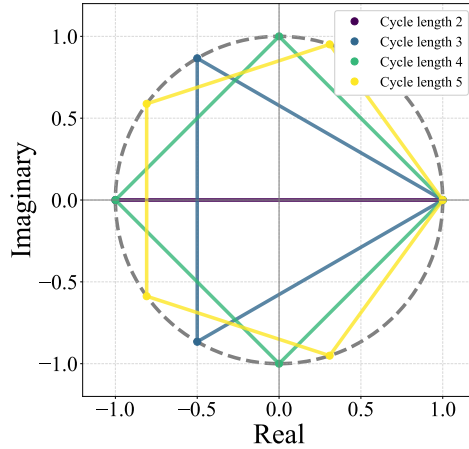

As elucidated Lezcano-Casado and Martınez-Rubio [40], the representation in $\mathcal{U} := \{A \in \mathbb{R}^{\frac{n(n-1)}{2}} \mid \mathrm{Im}\, \lambda_k(A - A^\top) \in (-\pi, \pi), \forall k\}$ of permutation matrices with an eigenvalue of $-1$ precisely lies on the boundary of $\mathcal{U}$. For problems where these permutation matrices serve as the optimal solution, it may cause the optimization path to deviate from $\mathcal{U}$. A straightforward method is to left-multiply the result of Equation (6) by a random orthogonal matrix $B \in \mathrm{SO}(n)$, thereby shifting the boundary of $\mathcal{U}$ to other eigenvalues. Theoretically, this left translation $L_B(O) := BO$ $(\forall O \in \mathrm{SO}(n))$ creates a diffeomorphism on $\mathrm{SO}(n)$ that transforms the representation of the permutation matrix $P$ in $\mathcal{U}$ from $\mathrm{logm}(P)$ to $\mathrm{logm}(B^\top P)$.

Figure 4: Eigenvalues corresponding to cycles of different lengths, where eigenvalues from the same cycle are connected to illustrate repeated values at $-1$.

To validate the efficacy of the aforementioned approach, we conducted a simple empirical study. Given a random orthogonal matrix $B$ with a determinant of 1, we randomly generate 1000 permutation

matrices $P$, where odd permutations are projected to $\mathrm{SO}(n)$ as done in Appendix D. We then estimate the probability of the eigenvalue $-1$ occurring in $P$ and $B^\top P$, respectively. We examine permutation matrices ranging in dimension $n$ from 3 to 50. Each experiment is replicated five times and the results are presented in Table 3. The findings show that the left translation $L_B$ effectively relocates the majority of permutation matrices' representations into the interior of $\mathcal{U}$.

Table 3: Probability (%) of eigenvalue $-1$ occurring in matrices.

| dimension $n$ | 3 | 5 | 10 | 20 | 50 |
|---|---|---|---|---|---|
| $P$ | 16.34 ±0.89 | 32.14 ±1.02 | 51.08 ±1.36 | 64.78 ±1.16 | 77.48 ±0.80 |
| $B^\top P$ | 15.92 ±1.29 | 16.42 ±1.18 | 8.36 ±0.67 | 5.94 ±0.29 | 8.66 ±1.12 |

*Remark* C.1. A high-level idea involves dynamically adjusting $B$ to ensure that the permutation matrix near the current iteration point consistently remains within the interior of $\mathcal{U}$. This is a special case of dynamic trivialization proposed in Lezcano Casado [39].

## D Theoretical details of the odd permutation

In this section, we provide the details of handling odd permutations and focus on elucidating the underlying theory.

To deal with the case where the permutation matrix $P$ corresponds to an odd permutation, our core idea is to find an agent, $\widehat{P}$, of $P$ within $\mathrm{SO}(n)$. In this way, we may move the orthogonal matrix $O \in \mathrm{SO}(n)$ to the vicinity of $\widehat{P}$, and then restore the result to the neighborhood of $P$. This idea is formalized by the following method:

1. We seek an agent of $P$ within $\mathrm{SO}(n)$, which is determined by solving the optimization problem:
$$\underset{\widehat{P} \in \mathrm{SO}(n)}{\arg\min} \|\widehat{P} - P\|_{\mathrm{F}}^2. \tag{20}$$
   The analytical solution to Equation (20) is given by $\widehat{P} = PD$, where $D = \mathrm{diag}(\{1, \ldots, 1, -1\})$ is the identity matrix with the last column multiplied by $-1$.

2. Given that $\widehat{P} \in \mathrm{SO}(n)$, we can, by substituting $P$ with $\widehat{P}$ in Equation (10), move the orthogonal matrix $O \in \mathrm{SO}(n)$ to the vicinity of $\widehat{P}$, yielding $\widehat{O}$.

3. By right-multiplying by $D^\top$, we map $\widehat{O}$ to the neighborhood of $P$, resulting in $\widetilde{O} = \widehat{O}D^\top$.

We now derive the analytical solution to Equation (20). By the definition of the Frobenius norm, we can express:
$$\begin{aligned}
\|\widehat{P} - P\|_{\mathrm{F}}^2 &= \mathrm{trace}((\widehat{P} - P)^\top(\widehat{P} - P)) \\
&= \mathrm{trace}(\widehat{P}^\top \widehat{P}) + \mathrm{trace}(P^\top P) - 2\,\mathrm{trace}(\widehat{P}^\top P) \\
&= 2n - 2\,\mathrm{trace}(\widehat{P}^\top P)
\end{aligned}$$

The last equation uses the fact that both $\widehat{P}$ and $P$ are orthogonal matrices, i.e., $\mathrm{trace}(\widehat{P}^\top \widehat{P}) = \mathrm{trace}(P^\top P) = n$. Considering the singular value decomposition (SVD) [33], we have $P = U\Sigma V^\top$, where $U$ and $V$ are orthogonal matrices, and $\Sigma$ is a diagonal matrix with non-negative real singular values on the diagonal. Since the orthogonal matrix singular value is 1, there exists $\Sigma = I$. By leveraging the cyclic property of the trace, we have:
$$\begin{aligned}
\mathrm{trace}(\widehat{P}^\top P) &= \mathrm{trace}(\widehat{P}^\top U \Sigma V^\top) \\
&= \mathrm{trace}(V^\top \widehat{P}^\top U) = \mathrm{trace}(Z),
\end{aligned} \tag{21}$$

where $Z := V^\top \widehat{P}^\top U$ is an orthogonal matrix with $\det Z = -1$. Hence, all elements $z_{i,j}$ of $Z$ satisfy $|z_{i,j}| \leq 1$ and $Z$ must have an odd number of $-1$ eigenvalues. Notice that the trace of a matrix is the sum of its elements on the main diagonal, Equation (21) is maximized when

$Z = \mathrm{diag}(\{1, 1, \ldots, 1, -1\})$. Thus, the analytical solution to Equation (20) is $\widehat{P} = (DP^\top)^\top = PD$, where $D = \mathrm{diag}(\{1, \ldots, 1, -1\})$ is the identity matrix with the last column multiplied by $-1$.

Since $D$ is an orthogonal matrix, right-multiplying by $D$ becomes a right translation $R_D(O) := OD$ ($\forall O \in \mathrm{O}(n)$). Notably, $\mathrm{O}(n)$ is equipped with a bi-invariant metric $\langle \cdot, \cdot \rangle_{\mathrm{F}}$, which implies that $R_D$ is an isometry, linking the the neighborhoods of $P$ and its agent $\widehat{P}$. Thus, we have $\langle AD, BD \rangle_{\mathrm{F}} = \langle A, B \rangle_{\mathrm{F}}$ for any $A, B \in \mathrm{O}(n)$. In other words, when $\widehat{O}$ is moved close enough to $\widehat{P}$, $\widetilde{O}$ will also be sufficiently close to $P$.

# E   More discussion on Step II

**Visualization.**   In Figure 5, we visualize the results of Equation (11) as the temperature parameter $\tau$ varies. The first row corresponds to the even permutation, while the second row corresponds to the odd permutation. The leftmost image ($\tau = 1.0$) represents the original orthogonal matrices $O$ from Step I, and the rightmost image ($\tau = 0.0$) is the permutation matrix $P$ that is closest to $O$. The temperature parameter $\tau$ controls how closely the resulting orthogonal matrices $\widetilde{O}$, obtained in Step II, approach $P$. As $\tau \to 0$, the resulting orthogonal matrices $\widetilde{O}$ increasingly converge to $P$.

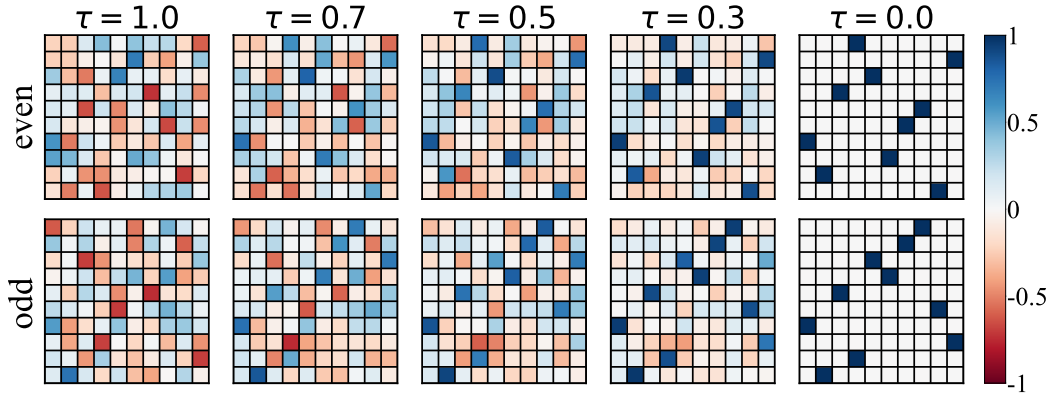

Figure 5: Visualization of the results of Equation (11) as the parameter $\tau$ varies. At $\tau = 1.0$, the matrices are original orthogonal matrices; at $\tau = 0.0$, they are the permutation matrices closest to original orthogonal matrices.

**Rounding to permutation matrix.**   Generally, the Hungarian algorithm requires a cost matrix without negative values, which is unsuitable for orthogonal matrices that may contain negative elements. To implement the rounding w.r.t. Equation (7) from the orthogonal matrix $O$, we first eliminate negative elements by subtracting the minimum element found within $O$, i.e., $O - \min(O)$. This approach is justified by the fact that $\arg\max_P \langle P, O \rangle_{\mathrm{F}} = \arg\max_P \langle P, O - \min(O) \rangle_{\mathrm{F}}$. Subsequent to this adjustment, we employ the Hungarian algorithm, available in existing libraries, to round $O - \min(O)$ to the closest permutation matrix.

**Computational costs.**   The primary computational costs of the proposed OT4P arise from solving the linear assignment problem and performing eigendecomposition, both of which typically scale with $\mathcal{O}(n^3)$. Numerous efforts have been made to accelerate these computations through parallel implementations on GPUs. We have employed existing implementations, specifically torch-linear-assignment [28, 29] and torch.linalg.eig [52].

# F Experimental details

## F.1 Reproducibility and compute resources

To enhance the reproducibility, we provide a comprehensive overview of the experiments in Appendices F.2 to F.4, including but not limited to data generation, hyperparameters, evaluation procedures. For all experiments, we repeat the results using five different random seeds: 2021, 2022, 2023, 2024, and 2025. The core code has been released at `https://github.com/YamingGuo98/OT4P`.

We conducted the experiments (Appendices F.2 and F.3) using a single **NVIDIA A800**, where the runtime environment was **Python=3.10**, **CUDA=11.7**, and **PyTorch=2.01**. The experiments in Section 4.1 took approximately 15 hours, while those in Section 4.2 consumed another 15 hours. This estimate does not include the time spent on hyperparameter search or other experiments conducted during the research process.

## F.2 Finding mode connectivity

**Steup.** We investigate various network architectures for Equation (17), including 5-layer MLP [54], VGG11 [60], and ResNet18 [22] For MLP5, both input and output are set to 1, with each of the four hidden layers having a dimension of 10, and the hyperbolic tangent serves as the activation function. The model weights of MLP5 are initialized randomly, while VGG11 and ResNet18 are derived from the official pre-trained models in PyTorch [52]. We minimize the loss corresponding to Equation (17) using AdamW [45] with an initial learning rate of 0.1 and a maximum of 500 iterations. Following Peña et al. [54], we adopt an early stopping strategy upon finding the optimal permutation matrices.

**Baseline.** We take `Weight Matching` [2] and `Sinkhorn` [54] as baselines. `Weight Matching` is a method specifically designed for addressing Equation (17), which goes through each layer and greedily selects its best permutation matrix $P_i$. We limit the maximum number of traversal rounds to 10. `Sinkhorn` relaxes permutation matrices to the vicinity of the Birkhoff polytope utilizing the Sinkhorn operator [61, 49], solving Equation (17) in a differentiable fashion. The Sinkhorn operator undergoes 20 iterations, with the temperature parameter set to 0.5.

**Evaluation.** During the evaluation phase, we round the matrices obtained from `Sinkhorn` and `OT4P` to permutation matrices using Equation (7). We employ the $\ell_1$-Distance, $\|\theta_A - \pi(\theta_B)\|_1$, to measure the difference in weights with the target network. In addition, we flatten the permutation matrices and evaluate their alignment with the ground truth using Precision, Recall, and Hamming Distance.

Table 4: Recall (%) and Hamming Distance of algorithms for finding mode connectivity across different network architectures.

| Algorithm | MLP5 | | VGG11 | | ResNet18 | |
|---|---|---|---|---|---|---|
| | Recall ($\uparrow$) | Hamming ($\downarrow$) | Recall ($\uparrow$) | Hamming ($\downarrow$) | Recall ($\uparrow$) | Hamming ($\downarrow$) |
| `Weight Matching` | 100.0 $_{\pm 0.00}$ | 0.000 $_{\pm 0.00}$ | 100.0 $_{\pm 0.00}$ | 0.000 $_{\pm 0.00}$ | 99.97 $_{\pm 0.06}$ | 1.600 $_{\pm 3.58}$ |
| `Sinkhorn` | 100.0 $_{\pm 0.00}$ | 0.000 $_{\pm 0.00}$ | 63.08 $_{\pm 3.14}$ | 2032 $_{\pm 173}$ | 95.56 $_{\pm 0.88}$ | 255.6 $_{\pm 50.8}$ |
| `OT4P` ($\tau = 0.3$) | 100.0 $_{\pm 0.00}$ | 0.000 $_{\pm 0.00}$ | 100.0 $_{\pm 0.00}$ | 0.000 $_{\pm 0.00}$ | 100.0 $_{\pm 0.00}$ | 0.000 $_{\pm 0.00}$ |
| `OT4P` ($\tau = 0.5$) | 100.0 $_{\pm 0.00}$ | 0.000 $_{\pm 0.00}$ | 99.99 $_{\pm 0.03}$ | 0.800 $_{\pm 1.79}$ | 100.0 $_{\pm 0.00}$ | 0.000 $_{\pm 0.00}$ |
| `OT4P` ($\tau = 0.7$) | 100.0 $_{\pm 0.00}$ | 0.000 $_{\pm 0.00}$ | 100.0 $_{\pm 0.00}$ | 0.000 $_{\pm 0.00}$ | 100.0 $_{\pm 0.00}$ | 0.000 $_{\pm 0.00}$ |

## F.3 Inferring neuron identities

**Setup.** Following the methodology in Linderman et al. [43], we generate parameters $A$, $W$, and $P$ with $n = 250$. Specifically, we randomly generate a binary upper triangular matrix $A \in \{0, 1\}^{n \times n}$ and its symmetric version $A = A + A^\top$ as the adjacency matrix. Parameter $W$ is sampled from a Gaussian distribution $\mathcal{N}(0, 1)$ according to the sparse sparsity defined by $A$. The permutation matrix $P$ is randomly sampled from a uniform distribution. We generate 1000 samples according to $Y_t = P(A \odot W)P^\top Y_{t-1} + \epsilon$, where the noise $\epsilon$ follows a Gaussian distribution $\mathcal{N}(0, 0.01)$.

We formulate tasks of varying difficulty depending on the different proportions of known neurons [43]. Conceiving a constraint matrix $C \in \mathbb{R}^{n \times n}$ where all elements are initialized to 1. If we ascertain that the reference neuron $i$ corresponds to the observed neuron $j$, then set all elements to 0 in the $i$-th row and $j$-th except for $C_{i,j}$. Equation (22) provides a simple example, where we know that the observed neuron 1 (or 3) corresponds to the reference neuron 2 (or 5).

$$
\begin{pmatrix} 1 & 1 & 1 & 1 & 1 \\ 1 & 1 & 1 & 1 & 1 \\ 1 & 1 & 1 & 1 & 1 \\ 1 & 1 & 1 & 1 & 1 \\ 1 & 1 & 1 & 1 & 1 \end{pmatrix} \xRightarrow{\text{know } (1,2)} \begin{pmatrix} 0 & 1 & 0 & 0 & 0 \\ 1 & 0 & 1 & 1 & 1 \\ 1 & 0 & 1 & 1 & 1 \\ 1 & 0 & 1 & 1 & 1 \\ 1 & 0 & 1 & 1 & 1 \end{pmatrix} \xRightarrow{\text{know } (3,5)} \begin{pmatrix} 0 & 1 & 0 & 0 & 0 \\ 1 & 0 & 1 & 1 & 0 \\ 0 & 0 & 0 & 0 & 1 \\ 1 & 0 & 1 & 1 & 0 \\ 1 & 0 & 1 & 1 & 0 \end{pmatrix}. \tag{22}
$$

This modeling decision significantly reduces the number of latent permutations that need to be inferred. The constraint matrix $C$ is enforced by zeroing corresponding entries, where for `Naive` it is before row normalization, for `Gumbel-Sinkhorn` it is before the Sinkhorn operator, and for `OT4P` it is before solving Equation (7).

**Baseline.** We compare `Naive` [43] and `Gumbel-Sinkhorn` [49]. `Naive` does not enforce $P$ to be a permutation matrix but instead normalizes each row using the softmax function. `Gumbel-Sinkhorn`, an extension of the Gumbel-Softmax [26, 46] method for permutations, introduces Gumbel noise for re-parameterization before the Sinkhorn operator. Both `Naive` and our `OT4P` use the stand Gaussian noise for re-parameterization. All methods estimate the gradient of the marginal log-likelihood $\mathbb{E}_{P \sim q(P;\theta)} \log p(Y|P)$ with 5 repeats.

**Evaluation.** We retain the best model throughout training, ranked first by Hamming Distance and then by the marginal log-likelihood (estimated with 5 repeats). We report the marginal log-likelihood of the best model. As done in Section 4.1, Precision, Recall, and Hamming Distance are utilized to evaluate the permutation matrices obtained from the best model without adding noise.

Table 5: Recall (%) and Hamming Distance of algorithms for inferring neuron identities across different proportions of known neurons.

| Algorithm | Known 5% | | Known 10% | | Known 20% | |
|---|---|---|---|---|---|---|
| | Recall ($\uparrow$) | Hamming ($\downarrow$) | Recall ($\uparrow$) | Hamming ($\downarrow$) | Recall ($\uparrow$) | Hamming ($\downarrow$) |
| Naive | $8.960_{\pm 7.85}$ | $455.2_{\pm 39.3}$ | $29.68_{\pm 17.2}$ | $351.6_{\pm 86.0}$ | $78.40_{\pm 12.6}$ | $108.0_{\pm 63.1}$ |
| Sinkhorn | $62.08_{\pm 16.0}$ | $189.6_{\pm 79.8}$ | $98.16_{\pm 1.95}$ | $9.200_{\pm 9.76}$ | $99.84_{\pm 0.358}$ | $0.800_{\pm 1.79}$ |
| OT4P ($\tau = 0.3$) | $100.0_{\pm 0.00}$ | $0.000_{\pm 0.00}$ | $100.0_{\pm 0.00}$ | $0.000_{\pm 0.00}$ | $100.0_{\pm 0.00}$ | $0.000_{\pm 0.00}$ |
| OT4P ($\tau = 0.5$) | $100.0_{\pm 0.00}$ | $0.000_{\pm 0.00}$ | $100.0_{\pm 0.00}$ | $0.000_{\pm 0.00}$ | $100.0_{\pm 0.00}$ | $0.000_{\pm 0.00}$ |
| OT4P ($\tau = 0.7$) | $74.16_{\pm 35.9}$ | $129.2_{\pm 180}$ | $100.0_{\pm 0.00}$ | $0.000_{\pm 0.00}$ | $100.0_{\pm 0.00}$ | $0.000_{\pm 0.00}$ |

### F.4 Solving permutation synchronization

**Setup.** We use the WILLOW-ObjectClass dataset [9] to generate problem instances. The WILLOW-ObjectClass dataset comprises images of five object classes, each containing 10 equal key points of at least 40 images. For each image, we extract interpolated features from the *relu4_2* and *relu5_1* layers through a pre-trained VGG16 [60] model on ImageNet [10]. The initial pairwise correspondences are established by applying the Hungarian algorithm [36] to the distance matrices of features. We increase the number of objectives, $k$, from 20 to the its largest value (multiples of 5) for each object class.

**Permutationness.** We also take the $\ell_1$-Distance to assess the 'permutationness' of the final matrix. Specifically, we round the matrix $\widetilde{P}$, returned by the algorithms, to its closest permutation matrix $P$, and then calculate the $\ell_1$-Distance between $\widetilde{P}$ and $P$. Table 6 lists the results for the problem instances corresponding to the largest size (multiples of 5) in each object class. We observe that the relaxation extent of `Sinkhorn` is unstable. Unlike them, `OT4P` consistently maintains smaller distances in almost all cases and exhibits a positive correlation with changes in the hyperparameter $\tau$.

Table 6: $\ell_1$-Distance between the matrix returned by the algorithms and its closest permutation matrix. In each object class, we select the largest problem instance size that is a multiple of five.

| Dataset | Reg | OrthReg | RiemanBirk | Sinkhorn | OT4P ($\tau = 0.3$) | OT4P ($\tau = 0.5$) | OT4P ($\tau = 0.7$) |
|---|---|---|---|---|---|---|---|
| Car | $13.13_{\pm 0.55}$ | $3.20_{\pm 0.90}$ | $10.91_{\pm 0.84}$ | $6.10_{\pm 0.06}$ | $2.30_{\pm 0.04}$ | $3.67_{\pm 0.16}$ | $4.68_{\pm 0.09}$ |
| Duck | $12.64_{\pm 0.83}$ | $3.26_{\pm 0.84}$ | $10.85_{\pm 0.33}$ | $5.65_{\pm 0.05}$ | $2.05_{\pm 0.02}$ | $3.15_{\pm 0.17}$ | $4.29_{\pm 0.08}$ |
| Face | $10.89_{\pm 0.44}$ | $3.01_{\pm 0.43}$ | $8.00_{\pm 0.46}$ | $0.09_{\pm 0.02}$ | $0.61_{\pm 0.01}$ | $0.98_{\pm 0.02}$ | $1.34_{\pm 0.01}$ |
| Motorbike | $15.17_{\pm 0.27}$ | $3.37_{\pm 0.89}$ | $13.30_{\pm 0.11}$ | $10.86_{\pm 0.02}$ | $3.10_{\pm 0.21}$ | $4.76_{\pm 0.29}$ | $5.90_{\pm 0.15}$ |
| Winebottle | $11.44_{\pm 0.41}$ | $3.17_{\pm 0.31}$ | $9.12_{\pm 1.13}$ | $2.62_{\pm 0.07}$ | $1.53_{\pm 0.02}$ | $2.36_{\pm 0.12}$ | $2.91_{\pm 0.12}$ |

**Impact of Optimizers.** We compare the results of different optimizers in Table 7, selecting the largest instances (multiples of 5) for each object class. Methods based on the Birkhoff polytope show notable performance improvements on most datasets when using (Riemannian) SGD. For our proposed OT4P, the choice of optimizer appears to be less critical, as it consistently outperforms other methods regardless.

Table 7: F-scores (%) for different algorithms with various optimizers on the WILLOW-ObjectClass dataset. In each object class, we select the largest problem instance size that is a multiple of five.

| Dataset | | Reg | OrthReg | RiemanBirk | Sinkhorn | OT4P ($\tau = 0.3$) | OT4P ($\tau = 0.5$) | OT4P ($\tau = 0.7$) |
|---|---|---|---|---|---|---|---|---|
| Car | SGD | $80.98_{\pm 1.68}$ | $78.52_{\pm 1.23}$ | $79.27_{\pm 3.07}$ | $81.43_{\pm 0.80}$ | $100.0_{\pm 0.00}$ | $100.0_{\pm 0.00}$ | $100.0_{\pm 0.00}$ |
| | Adam | $77.48_{\pm 3.15}$ | $92.76_{\pm 5.16}$ | $72.13_{\pm 3.45}$ | $82.94_{\pm 0.29}$ | $100.0_{\pm 0.00}$ | $98.61_{\pm 1.35}$ | $99.8_{\pm 0.40}$ |
| Duck | SGD | $90.84_{\pm 0.38}$ | $91.56_{\pm 2.46}$ | $87.14_{\pm 4.49}$ | $91.49_{\pm 0.06}$ | $100.0_{\pm 0.00}$ | $98.93_{\pm 0.58}$ | $99.63_{\pm 0.49}$ |
| | Adam | $75.52_{\pm 0.93}$ | $90.01_{\pm 5.21}$ | $69.89_{\pm 3.54}$ | $79.32_{\pm 0.23}$ | $100.0_{\pm 0.00}$ | $99.37_{\pm 1.26}$ | $98.71_{\pm 0.40}$ |
| Face | SGD | Failed | $100.0_{\pm 0.00}$ | $100.0_{\pm 0.00}$ | $100.0_{\pm 0.00}$ | $100.0_{\pm 0.00}$ | $100.0_{\pm 0.00}$ | $100.0_{\pm 0.00}$ |
| | Adam | $99.93_{\pm 0.12}$ | $91.10_{\pm 2.77}$ | $95.12_{\pm 2.55}$ | $100.0_{\pm 0.00}$ | $100.0_{\pm 0.00}$ | $100.0_{\pm 0.00}$ | $100.0_{\pm 0.00}$ |
| Motorbike | SGD | $50.00_{\pm 1.61}$ | $62.96_{\pm 1.23}$ | $49.76_{\pm 2.96}$ | $55.78_{\pm 1.31}$ | $98.51_{\pm 1.47}$ | $98.01_{\pm 2.25}$ | $98.99_{\pm 1.06}$ |
| | Adam | $45.57_{\pm 2.05}$ | $92.52_{\pm 4.68}$ | $43.72_{\pm 1.76}$ | $57.08_{\pm 0.75}$ | $97.22_{\pm 1.58}$ | $97.20_{\pm 2.04}$ | $99.61_{\pm 0.78}$ |
| Winebottle | SGD | $91.66_{\pm 0.31}$ | $91.51_{\pm 0.14}$ | $91.48_{\pm 0.00}$ | $91.44_{\pm 1.00}$ | $98.53_{\pm 1.48}$ | $98.65_{\pm 0.72}$ | $99.97_{\pm 0.03}$ |
| | Adam | $90.84_{\pm 0.38}$ | $91.56_{\pm 2.46}$ | $87.14_{\pm 4.49}$ | $91.49_{\pm 0.06}$ | $100.0_{\pm 0.00}$ | $98.93_{\pm 0.58}$ | $99.63_{\pm 0.49}$ |

**Case study.** This case study aims to explore the runtime and memory efficiency of our proposed OT4P on large-scale problems. We use the CMU House [5] image sequence to generate problem instances, comprising 111 frames of a video of a toy house. A total of $111 \times 111$ pairwise matching, with each image having 30 hand labeled landmark points, are provided in Bernard et al. [5]. We increase the number of objectives, $k$, from 20 to 110 and utilize the proposed OT4P to approximately solve Equation (18) on the **NVIDIA GeForce RTX 3090**. For all instances, we conduct 100 iterations using AdamW [45] with an initial learning rate of 0.1.

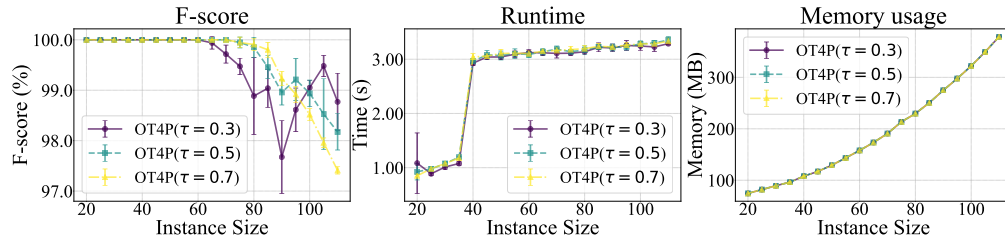

Figure 6: F-score, runtime, and memory usage of OT4P on the CMU House, where the size of permutation synchronization problem instances varies along the horizontal axis.

We flatten the permutation matrices and evaluate their alignment with the ground truth using F-score, while also assessing the runtime and the maximum GPU memory consumption during training. Each experiment is conducted five times, and the results are depicted in Figure 6, where the size of problem instances varies along the horizontal axis. The findings indicate that OT4P can find satisfactory solutions (with F-scores exceeding 97%) in less than 4 seconds, and the memory usage is manageable. Additionally, in Figure 7, we present the matching between the first and last images for $k = 110$, where the obtained matchings are connected (correct: ———, incorrect: ———). Overall, the

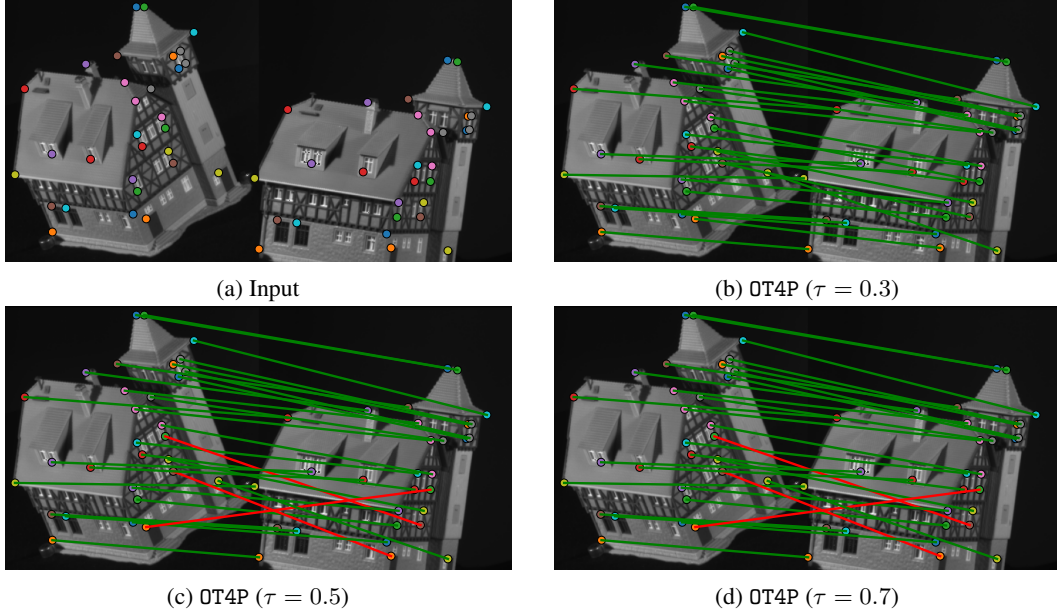

| (a) Input | (b) OT4P ($\tau = 0.3$) |
| (c) OT4P ($\tau = 0.5$) | (d) OT4P ($\tau = 0.7$) |

Figure 7: Matching between the first and last images of the CMU House for $k = 110$, where the obtained matchings are connected (green: correct, red: incorrect).

proposed OT4P demonstrates high efficiency in both runtime and memory usage when dealing with large-scale tasks amenable to parallel processing.

### F.5 Licenses

We will publicly release our code under the MIT license, in accordance with community standards. The licenses for the code, data, and models used in this study are provided below. Please refer to individual links for more details.

- **PyTorch:** BSD-style
- **Torchvision:** BSD 3-Clause
- **torch-linear-assignment:** Apache
- **sinkhorn-rebasin:** MIT
- **SparseStiefelOpt:** GNU Affero General Public
- **geoopt:** Apache

## G   Proof

In this section, we first elucidate some key properties of the matrix exponential and the matrix logarithm. Based on these, we prove Theorems 1 and 2 in Appendix G.2. Finally, we list some essential lemmas used in the proofs.

### G.1   Properties of matrix exponential and logarithm

**Theorem G.1.** *The matrix exponential map* $\mathrm{expm}\,(\cdot)$ *is injective over* $\mathcal{V} := \{A \in \mathbb{R}^{n \times n} \mid \mathrm{Im}\,\lambda_k(A) \in (-\pi, \pi),\ \forall k\}$.

*Proof.* For all $A, B \in \mathcal{V}$, our goal is to prove that $A = B$ if $\mathrm{expm}\,(A) = \mathrm{expm}\,(B)$.

According to Lemma G.1, if $\mathrm{expm}\,(A) = \mathrm{expm}\,(B)$, then $A$ and $B$ commute, i.e., $AB = BA$. In fact, this commutativity implies that the equation $\mathrm{expm}\,(A + B) = \mathrm{expm}\,(A)\,\mathrm{expm}\,(B)$ holds [23].

Thus, we can express

$$\mathrm{expm}\,(A) = \mathrm{expm}\,(B)$$
$$\mathrm{expm}\,(A)\,\mathrm{expm}\,(-B) = \mathrm{expm}\,(B)\,\mathrm{expm}\,(-B) = I$$
$$\mathrm{expm}\,(A - B) = I$$

The above equation shows that all eigenvalues of $\mathrm{expm}\,(A - B)$ are one. Since the eigenvalues of the exponential of a matrix are the exponential of its eigenvalues, the eigenvalues of $A - B$ satisfy $2m\pi i$ for integers $m \in \mathbb{Z}$. However, given that $A, B \in \mathcal{V}$, the range of $\mathrm{Im}\,\lambda_k(A - B)$ is restricted to $(-2\pi, 2\pi)$ for all $k$, yielding $m = 0$. Thus, all eigenvalues of $A - B$ are zero, which means $A - B$ is a nilpotent matrix.

Recalling Lemma G.2, we know that under the exponential map, the only nilpotent matrix mapped to the identity matrix $I$ is the null matrix. Therefore, we conclude that $A - B = \mathbf{0}$, which implies $A = B$.

This confirms that the matrix exponential map is injective within the set $\mathcal{V}$, thereby completing the proof. $\qquad\square$

**Theorem G.2.** *The matrix logarithm map $\mathrm{logm}\,(\cdot)$ is injective over $\mathcal{W} := \{A \in \mathbb{R}^{n\times n} \mid \lambda_k(A) \notin \mathbb{R}_0^-,\ \forall k\}$, and $\mathrm{logm}(\mathcal{W}) \subseteq \mathcal{V}$.*

*Proof.* We first prove that if $A \in \mathcal{W}$, then $B = \mathrm{logm}\,(A) \in \mathcal{V}$.

Assume $A \in \mathcal{W}$ with eigenvalues $\lambda_k = \rho_k e^{i\theta_k}$, where $\rho_k > 0$ and $\theta_k$ are real number. As shown in Gallier [18], the complex eigenvalues of $B = \mathrm{logm}\,(A)$ appear only for real Jordan blocks:

$$\begin{pmatrix} \log(\rho_k) & -\theta \\ \theta & \log(\rho_k) \end{pmatrix},$$

where the eigenvalues are given by $\log(\rho_k) \pm i\theta_k$. Since $A$ has no eigenvalues on $\mathbb{R}_0^-$, it follows that $\theta_k \in (-\pi, \pi)$ for all $k$. Thus, for any $A \in \mathcal{W}$, we have $\mathrm{logm}(A) \in \mathcal{V}$, that is, $\mathrm{logm}(\mathcal{W}) \subseteq \mathcal{V}$.

Recall that the matrix logarithm is defined as a solution to the equation $\mathrm{expm}\,(B) = A$. Coupled with Theorem G.1 and noting that $\mathrm{logm}(A) \in \mathcal{V}$, then such a logarithm is unique. This proves that the matrix logarithm map is injective over $\mathcal{V}$ and completes the proof. $\qquad\square$

**Corollary G.3.** *The image $\mathrm{expm}\,(\mathcal{V})$ of $\mathcal{V}$ by the matrix exponential $\mathrm{expm}\,(\cdot)$ is the set $\mathcal{W}$ and the mapping $\mathrm{expm} : \mathcal{V} \to \mathcal{W}$ is a diffeomorphism.*

*Proof.* By Theorem G.2, it is clear that $\mathcal{W} \subseteq \mathrm{expm}\,(\mathcal{V})$. We now prove $\mathrm{expm}\,(\mathcal{V}) \subseteq \mathcal{W}$.

For any matrix $A \in \mathcal{V}$, the eigenvalues have the form $a + ib$, where $a$ and $b$ are real number and $-\pi < b < \pi$. Then, the eigenvalues of $\mathrm{expm}\,(A)$ take the form $e^{a+ib} = e^a e^{ib}$. Since $e^{ib}$ never lies on $\mathbb{R}_0^-$, $\mathrm{expm}\,(A)$ has no non-positive real eigenvalues, i.e., $\mathrm{expm}\,(A) \in \mathcal{W}$. The arbitrariness of $A$ leads to the conclusion of $\mathrm{expm}\,(\mathcal{V}) \subseteq \mathcal{W}$.

The diffeomorphism results directly from Theorems G.1 and G.2. $\qquad\square$

### G.2 Proof of Theorems 1 and 2

**Theorem 1\*.** *The mapping $\phi(\cdot)$ is differentiable, surjective, and it is injective on the domain $\mathcal{U} := \{A \in \mathbb{R}^{\frac{n(n-1)}{2}} \mid \mathrm{Im}\,\lambda_k(A - A^\top) \in (-\pi, \pi),\ \forall k\}$ with $\lambda_k(\cdot)$ the eigenvalues. Additionally, the set $\mathrm{SO}(n) \setminus \phi(\mathcal{U})$ has a zero Lebesgue measure in $\mathrm{SO}(n)$.*

*Proof.* It is trivial that $\phi(\cdot)$ is differentiable.

We start by establishing that $\phi(\cdot)$ is surjective. It is worth mentioning that the conclusion of Lemma G.3 is applicable to $\mathrm{SO}(n)$ due to it being connected and compact. However, we provide a straightforward proof based on Theorem G.1. Indeed, any orthogonal matrix $O \in \mathrm{SO}(n)$ can be diagonalized into a block-diagonal matrix with diagonal blocks consisting of $2 \times 2$ rotation matrices:

$$\begin{pmatrix} \cos\theta & \sin\theta \\ -\sin\theta & \cos\theta \end{pmatrix},$$

where $\theta \in (-\pi, \pi]$ represents the rotation angle. For $\mathrm{SO}(2n + 1)$, an extra block containing a single 1 exists. Correspondingly, we have a skew-symmetric matrix $B \in \mathfrak{so}(n)$ satisfying $\mathrm{expm}\,(B) = O$, which is also a block-diagonal matrix with diagonal blocks of the form:

$$\begin{pmatrix} 0 & \theta \\ -\theta & 0 \end{pmatrix}.$$

There is an additional block with a single 0 on $\mathfrak{so}(2n + 1)$. As a result, for all orthogonal matrices $O \in \mathrm{SO}(n)$, we can surely find a skew-symmetric matrix $B \in \mathfrak{so}(n)$ such that $\mathrm{expm}\,(B) = O$. Given that $\mathfrak{so}(n)$ is isomorphism to the vector space $\mathbb{R}^{\frac{n(n-1)}{2}}$, this confirms that $\phi(\cdot)$ is surjective. Regarding $\phi(\cdot)$ is injective on the domain $\mathcal{U}$, it can be derived from Theorem G.1 because $\mathcal{U} \subseteq \mathcal{V}$.

The above reasoning also indicates that $\phi(\mathcal{U}) \subseteq \mathrm{SO}(n) \subseteq \phi(\bar{\mathcal{U}})$ holds and that the boundary of $\mathcal{U}$ has zero Lebesgue measure. Since the matrix exponential map $\mathrm{expm}\,(\cdot)$ is injective within the interior of $\mathcal{U}$ and the complex exponential function is single-valued, the set $\mathrm{SO}(n) \setminus \phi(\mathcal{U})$ has a zero Lebesgue measure in $\mathrm{SO}(n)$. $\qquad\square$

**Theorem 2\***. *The mapping $\psi_\tau(\cdot)$ is differentiable, surjective, and injective on each submanifold $\mathcal{S}_P$. Additionally, the set of meaningless points for $\psi_\tau(\cdot)$ has a zero Lebesgue measure in $\mathrm{SO}(n)$.*

*Proof.* Since the right translation $R_D$ establishes an isometry between the neighborhoods of $P$ and its agent $\widehat{P}$, we can, without loss of generality, restrict proof to the case where $P = \rho(O)$ is an even permutation. In this case, we rewrite $\psi_\tau$ as $\phi_\tau(O) = P\,\mathrm{expm}\,(\tau\,\mathrm{logm}\,(P^\top O))$. According to Corollary G.3, if $P^\top O$ is within $\mathcal{W}$, then we can assert that $\psi_\tau(\cdot)$ is surjective and injective on each submanifold $\mathcal{S}_P$.

Assuming the eigenvalues of $P^\top O$ are $\lambda_1, \ldots, \lambda_n$, the eigenvalues of $P^\top O - I$ are $\lambda_1 - 1, \ldots, \lambda_n - 1$. Since the inverse mapping of the left translation, $L_{P^\top}(O) := P^\top O$ ($\forall O \in \mathrm{SO}(n)$), maps $O$ to a neighborhood around $I$, we may assume $\|P^\top O - I\|_\mathrm{F} < 1$. A well-known result is $r(A) \leq \|A\|_\mathrm{F}$ for any $A \in \mathbb{R}^{n \times n}$, where $r(A) := \max\{|\lambda_k(A)| \mid \forall k\}$ is the spectral radius. As a result, we have:

$$r(P^\top O - I) = \max\{|\lambda_k - 1| \mid \forall k\} \leq \|P^\top O - I\|_\mathrm{F} < 1. \tag{23}$$

Equation (23) shows that the eigenvalues of $P^\top O$ are within a unit circle centered at 1 in the complex plane. Therefore, $P^\top O$ has no eigenvalues on $\mathbb{R}_0^-$, meaning $P^\top O \in \mathcal{W}$.

Since $\rho(\cdot)$ w.r.t. Equation (7) is a piecewise constant function, it follows that the set of points where $\psi_\tau(\cdot)$ is meaningless has a zero Lebesgue measure. This completes the proof. $\qquad\square$

### G.3  Lemmas used

**Lemma G.1** (Theorem 4. in Hille [25]). *Let $A, B \in \mathbb{C}^{n \times n}$. If there are no two eigenvalues in $A$ such that their difference is of the form $2k\pi i$ for $k > 0$ and, if $\mathrm{expm}\,(A) = \mathrm{expm}\,(B)$, we have that $AB = BA$.*

Let $\mathrm{Nil}(k)$ denote the set of (real or complex) nilpotent matrices, $A$, of any dimension $n \geq 1$ such that $A^r = \mathbf{0}$ and $\mathrm{Uni}(k)$ denote the set of unipotent matrices, $B = I + A$, where $A \in \mathrm{Nil}(k)$.

**Lemma G.2** (Proposition 3.2. in Gallier [18]). *The map $\mathrm{expm} : \mathrm{Nil}(k) \to \mathrm{Uni}(k)$ is a homeomorphism whose inverse is the matrix logarithm.*

**Lemma G.3** (Corollary 11.10. in Hall [21]). *If $G$ is a connected, compact matrix Lie group, the exponential map for $G$ is surjective.*

